# Multi-Winner Reconfiguration

**Jiehua Chen**
Institute of Logic and Computation
TU Wien
Austria
jchen@ac.tuwien.ac.at

**Christian Hatschka**
Institute of Logic and Computation
TU Wien
Austria
chatschka@ac.tuwien.ac.at

**Sofia Simola**
Institute of Logic and Computation
TU Wien
Austria
ssimola@ac.tuwien.ac.at

## Abstract

We introduce a multi-winner reconfiguration model to examine how to transition between two subsets of alternatives (aka. *committees*) through a sequence of minor yet impactful modifications, called *reconfiguration path*. We analyze this model under four approval-based voting rules: Chamberlin-Courant (CC), Proportional Approval Voting (PAV), Approval Voting (AV), and Satisfaction Approval Voting (SAV). The problem exhibits computational intractability for CC and PAV, and polynomial solvability for AV and SAV. We provide a detailed multivariate complexity analysis for CC and PAV, demonstrating that although the problem remains challenging in many scenarios, there are specific cases that allow for efficient parameterized algorithms.

## 1   Introduction

In the rapidly evolving landscape of technology, policy, and market dynamics, we need to be able to adapt and reconfigure existing solutions. This necessity spans various domains, such as disaster management (Ito *et al.*, 2011; Ju *et al.*, 2017) – where one needs to switch, step by step, from an existing power supply distribution to another one, while ensuring no blackouts occur in-between – and game and puzzle-solving (Nishimura, 2018) – where one needs to transfer from one feasible state to another via a sequence of feasible moves. The crucial idea is to employ a systematic approach to transition from one solution to another with minimal disruption. Such a concept, known as *reconfiguration* (Ito *et al.*, 2011), is omnipresent in areas ranging from everyday life to complex decision-making processes in computational social choice (COMSOC) such as switching catalogs in streaming services (Prime-Video, 2024; Netflix, 2024) or product displays in department stores.

This paper delves into reconfiguration within COMSOC, using multi-winner elections as a primary example to illustrate its importance and applicability. Our focus is on how to find such a reconfiguration through incremental yet meaningful changes, adhering to a set criterion that quantifies the "closeness" between successive solutions. In multi-winner elections, we are given a preference profile consisting of a finite set of alternatives and a finite set of voters, each with preferences over the alternatives. The objective is to select a *committee* (i.e., a subset of alternatives) of fixed size k, representing voters' preferences optimally. Briefly put, in *multi-winner reconfiguration* we need to decide whether there exists a transformation between two size-k (not necessarily optimal) committees via a sequence of intermediate size-k committees that are "almost" as good as the input committees

and close to one another (see Section 2 for the formal definition). Multi-winner reconfiguration can model many real-world applications:

– Streaming service providers, such as Prime Video and Netflix, change their catalog frequently (Prime-Video, 2024; Netflix, 2024) to keep it engaging, while ensuring that popular or critically acclaimed content remains available. Here, customer *satisfaction* hinges on a balanced mix of fresh and familiar content. The multi-winner reconfiguration model can be used to optimize this balance, ensuring the catalog evolves in a way that minimizes disruption to viewer experience.
– Likewise, for physical or online stores, product displays are crucial in attracting and retaining customers. However, generally shelf or display space is limited. The challenge is to introduce new products or promotions without overshadowing popular or staple items. The reconfiguration model can guide the arrangement of products using fixed capacity in a way that maximizes visibility for new items while ensuring customer satisfaction.
– An online gaming platform may need to shut down some old games to free up server space to support new games. However, abruptly removing a lot of games could alienate players, and thus the changes should be implemented incrementally in order to ensure player satisfaction throughout the whole exchange process.

In each of the above scenarios, the goal revolves around transitioning from one committee (i.e., a subset of products, movies, games, or routes) to another through minimal, incremental changes, ensuring a balance between introducing new elements and maintaining overall user satisfaction. In other words, we need to ensure that the consecutive committees in the transition are close to each other, both in their content and optimality wrt. the initial committee.

**Our contributions.** We propose a model of multi-winner reconfiguration and study its computational complexity. We focus on four multi-winner voting rules for approval preferences: *Chamberlin-Courant* (CC), *Proportional Approval Voting* (PAV), *Approval Voting* (AV), and *Satisfaction Approval Voting* (SAV). We find that while for the first two rules determining whether a reconfiguration path exists is PSPACE-complete, for the latter two a reconfiguration path always exists. We further investigate the parameterized complexity for CC and PAV by looking at several canonical parameters: the number $m$ of alternatives, the number $n$ of voters, the committee size $k$, the maximum number $b$ of approved alternatives per voter, and the length $\ell$ of a shortest reconfiguration path (if it exists). The parameters $n$, $m$ and $k$ are natural input parameters that are already studied in the original multi-winner determination problem. The parameter b can be considered as a structural and distance to tractability parameter: For $b = 1$, all studied voting rules are equivalent to AV, so by Theorem 1, the reconfiguration problem can be solved in polynomial time. Thus, it is natural to ask whether the problem is polynomial-time solvable for each constant b. Finally, by our reduction, the PSPACE-hardness arises because a shortest reconfiguration path could be of unbounded length. Note that for bounded length $\ell$ the problem is in NP, see Proposition 2. In real-world applications, one may need to reconfigure committees timely, so it is natural to assume that $\ell$ is small. We summarize our findings as follows (also see Table 1); For a brief definition of the relevant parameterized complexity classes see Section 2:

(i) The problem is fixed-parameter tractable (FPT) wrt. $m$. Under CC it is also FPT wrt. $n$.
(ii) While the problem is in XP wrt. $n$ (resp. k). it remains NP-hard even for constant value $b + \ell$. The XP result for k is essentially tight since it remains W[2]-hard under CC (resp. W[1]-hard under PAV).
(iii) Combining $n$ with any other single parameter always yields an FPT algorithm.

Due to space constraints, proofs of the results and additional materials marked with ($\star$) are deferred to the appendix.

**Related Work.** Reconfiguration of puzzles and games has been of interest to mathematicians already since the 19th century (Johnson and Story, 1879). Since the early 2000s, reconfiguration has attracted increasing attention in the fields of computational geometry (Connelly *et al.*, 2003; Aichholzer *et al.*, 2015; Alamdari *et al.*, 2017), graph theory (Ito *et al.*, 2011; Bonsma, 2013), constraint satisfaction (Gopalan *et al.*, 2006), and quantum complexity theory (Gharibian and Sikora, 2018). For graph problems, there are in general three types of reconfigurations between feasible solutions: token addition and removal (TAR; in each step one can either remove or add a vertex), token jumping (TJ; in each step one can replace a vertex with another vertex), and token sliding (TS; in each step one can replace a vertex with another adjacent vertex). Our multi-winner reconfiguration problem generalizes the TJ model in the sense that we allow replacement of more than one alternative

Table 1: (Parameterized) complexity results obtained in our paper. We omit the results for AV and SAV as reconfiguration under both rules is polynomial-time solvable (see Theorem 1). Both PSPACE-hardness and W[1]-hardness hold even if each voter approves of at most two alternatives, i.e., $b = 2$. Except for the NP-hardness for constant $b + \ell$, all hardness results hold even if $(\delta_c, \delta_s) = (1, 0)$.

| Parameter | CC | | PAV | |
|---|---|---|---|---|
| In general | PSPACE-c | [T2] | PSPACE-c | [T2] |
| $m$ | FPT | [P3] | FPT | [P3] |
| $n$ | FPT | [T4] | ?, XP | –,[P4] |
| k | W[2]-h, XP | [T5], [P3] | W[1]-h, XP | [T2], [P3] |
| $b + \ell$ | NP-h | [P2] | NP-h | [P2] |
| $k + \ell$ | W[2]-h, XP | [T5], [P3] | W[1]-h, XP | [T2], [P3] |
| $b + k + \ell$ | W[1]-h | [T2] | W[1]-h | [T2] |
| $n + b$ | FPT | [T4] | FPT | [P5] |
| $n + k$ | FPT | [T4] | FPT | [P4] |
| $n + \ell$ | FPT | [T4] | FPT | [T3] |

in each step. For a general introduction to reconfiguration and how it differs from various similar problems such as local search (Ahuja *et al.*, 2002), reoptimization (Schieber *et al.*, 2018), and incremental problems (Bredereck *et al.*, 2020a), we refer to the surveys by van den Heuvel (2013) and Nishimura (2018). Ito *et al.* (2014), Mouawad *et al.*. (2016), and Bodlaender *et al.* (2021) studied the parameterized complexity of various reconfiguration problems such as independent set and dominating set reconfiguration. They considered the length of the path as well as a bound on the size of the solutions as parameters. Notably, Bodlaender et al. claimed that dominating set reconfiguration via TJ is W[2]-hard wrt. the solution size and the reconfiguration path by wrongly referring to the TAR model. This is in general not correct as the two models are not equivalent. We correct this by providing a direct parameterized reduction from the dominating set problem (see Proposition 6).

Reconfiguration in computational social choice as a topic has recently gained traction, with Igarashi *et al.* (2024) introducing the concept of reconfiguration to fair division. Ito *et al.* (2022) introduced a similar framework to reconfiguration to fair matchings.

Multi-winner elections are a topic of great interest in computational decision theory. For a general introduction to multi-winner elections, see for example the work of Faliszewski *et al.* (2017). For further methods and topics on multi-winner elections with approval preferences we refer to a recent book by Lackner and Skowron (2023). In this work, we will only consider approval preferences.

Reconfiguration is related to the model of "one problem, successive solutions" by Boehmer and Niedermeier (2021) where the goal is to select a sequence of desirable solutions. Bredereck *et al.* (2020c) introduced and studied this model for multi-winner election where each alternative in the committee must remain in office a consecutive amount of times. Our reconfiguration problem is different from theirs in two ways: First, we aim at a sequence of committees starting from an initial committee and ending at a target committee such that the intermediate committees are "close" to each other. Second, we do not require that an alternative appears for a limited number of times.

## 2 Preliminaries

Given a non-negative integer $t \in \mathbb{N}$, let $[t]$ denote the set $\{1, \ldots, t\}$. Given two subsets $X$ and $Y$, we use $X \Delta Y$ to denote the symmetric difference between $X$ and $Y$, i.e., $X \Delta Y = (X \setminus Y) \cup (Y \setminus X)$.

**Parameterized Complexity.** Let $\Pi$ denote a decision problem and $k$ a parameter. We say that $\Pi$ is *fixed-parameter tractable* (*FPT*) wrt. $k$ if there exists an algorithm that decides every instance $I$ of $\Pi$ in $f(k) \cdot |I|^{O(1)}$ time (we say the algorithm runs in *FPT time*), where $f$ is a computable function that solely depends on $k$. We say that $\Pi$ is in *XP* wrt. $k$ if there exists an algorithm that decides every instance $I$ of $\Pi$ in $|I|^{f(k)}$ time, where $f$ is a computable function that solely depends on $k$. In the same way NP-hard problems are unlikely to be in P, problems which are *W[1]-* or *W[2]-hard* wrt. $k$ are unlikely to be FPT wrt. $k$. Similarly to showing NP-hardness, we show W[1]- and W[2]-hardness through FPT time reductions, where the new parameter must be a function of the original one.

As examples, INDEPENDENT SET wrt. solution size and DOMINATING SET wrt. solution size are W[1]-hard and W[2]-hard, respectively. For more details we refer to Cygan *et al.* (2015).

**Approval profiles and types.** An *approval preference profile* (*profile* for short) is a triple $\mathcal{P} = (C, V, R)$, where $C$ denotes a set of $m$ alternatives, $V$ a set of $n$ voters with $V = \{v_1, \ldots, v_n\}$, and $R$ a collection $R = (\mathsf{A}_1, \ldots, \mathsf{A}_n)$ of *non-empty* subsets of $C$ such that $\mathsf{A}_i$ consists of all alternatives that voter $v_i$ approves of, $i \in [n]$. Unless stated otherwise, we use $\mathcal{P}$ to denote a profile of the form $(C, V, R)$. We say that two alternatives $a$ and $b$ are of the same *type* if they are approved by the same voters. We call any subset of alternatives a *committee*.

**Score-based multi-winner voting rules.** A score-based (multi-winner) *voting rule* $\lambda$ (*voting rule* for short) takes as input a profile $\mathcal{P}$ and a number k, and assigns to each size-k committee $W$ a score $\mathrm{SC}_\lambda(\mathcal{P}, W)$. The committees with maximum score are *winning committees*. The score $\mathrm{SC}_\lambda(\mathcal{P}, W)$ measures the satisfaction of the voters towards committee $W$. Note that we will omit the profile $\mathcal{P}$ and the subscript $\lambda$ if it is clear from the context. We say that a voting rule is *neutral* if the outcome does not depend on the names of the alternatives. This implies that if two alternatives are of the same type, then replacing one alternative with the other does not change the outcome. We consider four *score-based voting rules*, all of which are neutral:

**The *Chamberlin-Courant* (CC)** rule selects a size-k committee that maximizes the number of voters who approve of at least one alternative in the committee. Formally, $W$ is winning under CC if it maximizes $\mathrm{SC}_{\mathrm{CC}}(W) = |\{v_i \in V : \mathsf{A}_i \cap W \neq \emptyset\}|$.

**The *Proportional Approval Voting* (PAV)** rule uses $h(x) = \sum_{j=1}^{x} \frac{1}{j}$ to denote the partial sum of the first $x$ elements of the harmonic sequence. A size-k committee $W$ is winning under PAV if it maximizes $\mathrm{SC}_{\mathrm{PAV}}(W) = \sum_{v_i \in V} h(|\mathsf{A}_i \cap W|)$.

**The *Approval Voting* (AV)** (resp. ***Satisfaction Approval Voting* (SAV)**) rule selects a size-k committee that maximizes the sum of approving voters (resp. the weighted sum of approving voters). Formally, the score of an alternative $a$ is defined as $\mathrm{SC}_{\mathrm{AV}}(a) = |\{v_i \in V \mid a \in A_i\}|$ (resp. $\mathrm{SC}_{\mathrm{SAV}}(a) = \sum_{v_i \in V \text{ s.t. } a \in A_i} \frac{1}{|\mathsf{A}_i|}$). Committee $W$ is winning under AV (resp. SAV) if it maximizes $\mathrm{SC}_{\mathrm{AV}}(W) = \sum_{a \in W} \mathrm{SC}_{\mathrm{AV}}(a)$ (resp. $\mathrm{SC}_{\mathrm{SAV}}(W) = \sum_{a \in W} \mathrm{SC}_{\mathrm{SAV}}(a)$).

**Reconfiguration of committees.** Let $\mathcal{P}$ be a profile and $\lambda$ a voting rule. Given a number $d$, we say that two committees $W$ and $W'$ are *d-adjacent* if $|W \setminus W'| \leq d$ and $|W' \setminus W| \leq d$. For two numbers $\delta_{\mathrm{c}} \geq 1$ and $\delta_{\mathrm{s}} \geq 0$, we call a sequence of committees $W_0, \ldots, W_t$ a $(\delta_{\mathrm{c}}, \delta_{\mathrm{s}})$-*reconfiguration path* (*reconfiguration path* for short) for $(\mathcal{P}, \lambda)$ if for all $i \in [t]$ the following holds:

(C1) $|W_{i-1}| = |W_i|$;
(C2) $W_{i-1}$ and $W_i$ are $\delta_{\mathrm{c}}$-adjacent;
(C3) The score difference of $W_0$ and $W_i$ is bounded by $\delta_{\mathrm{s}}$, i.e., $\mathrm{SC}_\lambda(W_0) - \mathrm{SC}_\lambda(W_i) \leq \delta_{\mathrm{s}}$.

**Remark 1.** Note that the reconfiguration path requires closeness (C2–C3) between committees of the same size (C1). This is essential, as closeness captures the incremental yet necessary changes necessary in various applications, like evolving content in streaming services or adjusting product line-ups in retail stores, while fixing the committee size (C1) mirrors practical limitations like pre-set numbers of movie slots in streaming catalogs or specific shelf capacities in retail stores.

By convention, the *length* of a reconfiguration path is defined as the number of committees on the path minus one.

**Example 1.** *Let $C = [4]$ and $V = \{v_1, v_2, v_3, v_4\}$ with approval preferences $\mathsf{A}_1 = \{1, 2\}, \mathsf{A}_2 = \{2, 3\}, \mathsf{A}_3 = \{3, 4\}$ and $\mathsf{A}_4 = \{1, 4\}$. Let k $= 2$. Under both CC and PAV we have two size-2 committees with maximum score of $4$: $W = \{1, 3\}$ and $W' = \{2, 4\}$. For $\delta_{\mathrm{c}} = \delta_{\mathrm{s}} = 1$ and under CC, there is a $(\delta_{\mathrm{c}}, \delta_{\mathrm{s}})$-reconfiguration path between $W$ and $W'$ of length two, e.g., $(W, \{1, 2\}, W')$. The same path is also a $(1, \frac{1}{2})$-reconfiguration path under PAV since $\mathrm{SC}_{\mathrm{PAV}}(\{1, 2\}) = 3\frac{1}{2}$.*

**Problem definition.** Now, we define our reconfiguration problem.

$\lambda$-MULTI-WINNER RECONFIGURATION ($\lambda$-MR)
**Input:** A profile $\mathcal{P}$, a committee size k, two numbers $\delta_{\mathrm{c}} \geq 1$ and $\delta_{\mathrm{s}} \geq 0$, and two size-k committees $W \neq W'$ with $\mathrm{SC}_\lambda(W) - \mathrm{SC}_\lambda(W') \leq \delta_{\mathrm{s}}$.
**Question:** Does there exist a sequence $(W_0, \ldots, W_t)$ of committees which is a $(\delta_{\mathrm{c}}, \delta_{\mathrm{s}})$-reconfiguration path for $(\mathcal{P}, \lambda)$ such that $W_0 = W$ and $W_t = W'$?

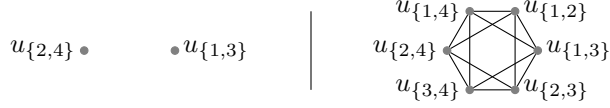

Figure 1: Reconfiguration graphs for Example 2 with $\delta_s = 0$ (Left), and with $\delta_s = 1$ under CC or $\delta_s = \frac{1}{2}$ under PAV (Right).

**Remark 2.** Note that we require that the input satisfies $\delta_c \geq 1$, $W \neq W'$, and $\mathrm{sc}_\lambda(W) - \mathrm{sc}_\lambda(W') \leq \delta_s$, as otherwise either there is always a reconfiguration path (when $W = W'$) or none (when $\delta_c < 1$ and $W \neq W'$, or $\mathrm{sc}_\lambda(W) - \mathrm{sc}_\lambda(W') > \delta_s$). Moreover, we intentionally do not restrict the committees to be optimal (i.e., winning). This choice allows for a more realistic representation of scenarios where optimal solutions are not immediate or feasible, thereby enhancing the practical relevance of our study.

For the restricted variant where the committees on the reconfiguration path must be optimal (i.e., $W$ and $W'$ are winning and $\delta_s = 0$) and $\delta_c = 1$, we write $\lambda\text{-}\mathrm{MR}^0$. We will see that all intractability results except Proposition 2 already hold for the restricted case while all algorithmic results hold for the general case.

**Reconfiguration graphs.** Let $I = (\mathcal{P}, \mathsf{k}, \delta_c, \delta_s, W, W')$ be an instance of $\lambda\text{-MR}$ with $\mathcal{P} = (C = [m], V, R)$ being a profile and $\lambda$ a score-based voting rule. We construct a *reconfiguration graph* $G^{\mathsf{R}} = (V^{\mathsf{R}}, E^{\mathsf{R}})$ for $I$ as follows. Let $V^{\mathsf{R}} = \{u_{C'} \mid C' \subseteq C \text{ s.t. } |C'| = \mathsf{k} \text{ and } \mathrm{sc}_\lambda(\mathcal{P}, W) - \mathrm{sc}_\lambda(\mathcal{P}, C') \leq \delta_s\}$ be the vertex set of $G^{\mathsf{R}}$ such that each vertex corresponds to a size-$\mathsf{k}$ committee whose score is at most $\delta_s$ lower than the score of $W$. For each pair $u_{W_1}, u_{W_2}$ of vertices , we create an edge $\{u_{W_1}, u_{W_2}\}$ if the corresponding two committees $W_1$ and $W_2$ are $\delta_c$-adjacent.

**Remark 3.** Note that $|V^{\mathsf{R}}| \leq \binom{m}{\mathsf{k}}$ since only committees of size $\mathsf{k}$ are relevant. We can compute $V^{\mathsf{R}}$ in time $\binom{m}{\mathsf{k}} \cdot \mathsf{T}(\mathcal{P}, \lambda)$, where $\mathsf{T}(\mathcal{P}, \lambda)$ denotes the time to compute the score of a committee under rule $\lambda$. The number of edges of $G^{\mathsf{R}}$ is bounded above by $\binom{m}{\mathsf{k}}^2$, so the reconfiguration graph can be constructed in time $O(\binom{m}{\mathsf{k}}^2 + \binom{m}{\mathsf{k}} \cdot \mathsf{T}(\mathcal{P}, \lambda))$. By construction, if $\delta_s = 0$, and both $W$ and $W'$ are winning committees, then for each pair of connected vertices $u_{W_1}$ and $u_{W_2}$, there exists a one-to-one correspondence between all $(u_{W_1}, u_{W_2})$-paths in $G^{\mathsf{R}}$ and all $(\delta_c, \delta_s)$-reconfiguration paths for $W_1$ and $W_2$.

**Example 2.** *Let us consider the instance given in Example 1. If $\delta_s = 0$, then the vertex set of the reconfiguration graph $G^{\mathsf{R}}$ is $V^{\mathsf{R}} = \{u_{\{1,3\}}, u_{\{2,4\}}\}$ since $\{1,3\}$ and $\{2,4\}$ are the only two winning committees. Because $|\{1,3\}\Delta\{2,4\}| = 4$, if $\delta_c = 1$, there is no path between $u_W$ and $u_{W'}$ and hence no reconfiguration path from $W$ to $W'$ exists. See the LHS of Figure 1 for an illustration.*

*If $\delta_s = 1$ (under CC) or $\delta_s = \frac{1}{2}$ (under PAV), then $V^{\mathsf{R}}$ contains vertices for all size-2 committees of $C$. There are four possible length-2 paths between $u_{\{1,3\}}$ and $u_{\{2,4\}}$, for example $(u_{\{1,3\}}, u_{\{1,2\}}, u_{\{2,4\}})$. See the RHS of Figure 1 for an illustration. Note that the length of a shortest path between any two committees $W, W'$ is at least $\left\lceil \frac{|W \Delta W'|}{2\,\delta_c} \right\rceil$; any edge on the reconfiguration will remove (resp. add) at most $\delta_c$ alternatives from $W \setminus W'$ (resp. $W' \setminus W$).*

## 3   Complexity Results

In this section, we consider the computational and parameterized complexity of $\lambda\text{-MR}$.

**Computational Complexity.** It is straightforward that $\lambda\text{-MR}$ is in PSPACE due to an observation by Ito *et al.* (2011).

**Proposition 1.** *For each $\lambda$ of the four considered voting rule, $\lambda\text{-MR}$ is in PSPACE.*

*Proof.* It suffices to show that $\lambda\text{-MR}$ is contained in NPSPACE since PSPACE = NPSPACE (Savitch, 1970). Moreover, as observed by Ito *et al.* (2011), to show that the reconfiguration version of a problem is in NPSPACE, one only needs to show that the underlying problem is in NP. In our case, we need to show that the following two problems can be verified in polynomial time: (1) Does a

given committee satisfy a given score-bound? (2) Are two given committees $\delta_c$-adjacent? For the sake of completeness, we provide a detailed proof for $\lambda$-MR. We first define a decision variant of the multi-winner determination problem. In $\lambda$-MULTI-WINNER SCORE, we are given a profile $\mathcal{P}$, a committee size k, and a number $s$, and ask whether there exists a size-k committee $W$ such that $\text{SC}_\lambda(W) \geq s$.

Note that since the score of a committee under each considered rule $\lambda$ can be computed in polynomial time, $\lambda$-MULTI-WINNER SCORE is in NP. Since NP $\subseteq$ PSPACE, given an instance $(\mathcal{P}, W, W', \delta_c, \delta_s)$ of $\lambda$-MR we can enumerate all size-k committees $U$ with $\text{SC}_\lambda(\mathcal{P}, W) - \text{SC}_\lambda(\mathcal{P}, U) \leq \delta_s$ in PSPACE. Then, we can iteratively non-deterministically choose a size-k committee $W''$ that is $\delta_c$-adjacent to the current committee and has $\text{SC}_\lambda(\mathcal{P}, W) - \text{SC}_\lambda(\mathcal{P}, W'') \leq \delta_s$. The problem is thus in NPSPACE. $\quad\square$

The following result implies that AV- and SAV-MR are easy. This is because both rules are based on *additive* scoring functions, so we can greedily exchange alternatives from the symmetric difference between the initial and target committees.

**Theorem 1.** *Every instance of* AV-MR *(resp.* SAV-MR*) admits a reconfiguration path. Moreover, a shortest one can be computed in polynomial time and has length* $\lceil \frac{|W \setminus W'|}{\delta_c} \rceil = \lceil \frac{|W \Delta W'|}{2\,\delta_c} \rceil$.

*Proof.* Let $\lambda \in \{\text{AV}, \text{SAV}\}$ and $I = (\mathcal{P}, k, \delta_s, \delta_c, \lambda, W, W')$ denote an instance of $\lambda$-MR. As already discussed, we will greedily exchange alternatives from $W \Delta W'$. The correctness is based on the following claims:

**Claim 1.1.** *Let $W_1$ and $W_2$ be two size-k committees, and $a = \arg\max_{c \in W_2 \setminus W_1} \text{SC}_\lambda(c)$ and $b = \arg\min_{c \in W_1 \setminus W_2} \text{SC}_\lambda(c)$. Then, $\text{SC}_\lambda((W_1 \cup \{a\}) \setminus \{b\}) \geq \min\{\text{SC}_\lambda(W_1), \text{SC}_\lambda(W_2)\}$.*

*Proof.* We distinguish between two cases. Case (1) $\text{SC}_\lambda(a) \geq \text{SC}_\lambda(b)$; Case (2) $\text{SC}_\lambda(a) < \text{SC}_\lambda(b)$.

For Case (1), due to the additivity of the voting rule the statement follows: $\text{SC}_\lambda(W_1 \cup \{a\} \setminus \{b\}) = \text{SC}_\lambda(W_1) + \text{SC}_\lambda(a) - \text{SC}_\lambda(b) \geq \text{SC}_\lambda(W_1) \geq \min\{\text{SC}_\lambda(W_1), \text{SC}_\lambda(W_2)\}$.

For Case (2), by the choice of $a$ and $b$, we infer that $\text{SC}_\lambda(c) < \text{SC}_\lambda(d)$ for all $c \in W_2 \setminus W_1$ and all $d \in W_1 \setminus W_2$. We now enumerate the elements in $W_2 \setminus W_1$ in the order $a_1, \ldots, a_t$ so that $a_i \geq a_{i+1}$ holds for all $i \in [t-1]$. Similarly, we enumerate $W_1 \setminus W_2$ in the order $b_1, \ldots, b_t$ so that $b_i \geq b_{i+1}$ holds for all $i \in [t-1]$. Note that in these orderings we have $a = a_1$ and $b = b_t$. Using this enumeration we can now show the statement:

$$
\begin{aligned}
& \text{SC}_\lambda(W_1 \cup \{a\} \setminus \{b\}) = \text{SC}_\lambda(W_1) + \text{SC}_\lambda(a) - \text{SC}_\lambda(b) \\
=\ & \text{SC}_\lambda(W_1 \cap W_2) + \textstyle\sum_{i \in [t]} \text{SC}_\lambda(b_i) + \text{SC}_\lambda(a) - \text{SC}_\lambda(b) \\
=\ & \text{SC}_\lambda(W_1 \cap W_2) + \textstyle\sum_{i \in [t-1]} \text{SC}_\lambda(b_i) + \text{SC}_\lambda(a) \\
\geq\ & \text{SC}_\lambda(W_1 \cap W_2) + \textstyle\sum_{i \in [t-1]} \text{SC}_\lambda(a_i) + \text{SC}_\lambda(a) \\
\geq\ & \text{SC}_\lambda(W_1 \cap W_2) + \textstyle\sum_{i \in [t]} \text{SC}_\lambda(1_i) = \text{SC}_\lambda(W_2) \qquad\qquad \text{(of Claim 1.1)} \diamond
\end{aligned}
$$

By Claim 1.1, we can iteratively replace the lowest scoring alternative in $W \setminus W'$ with the highest scoring alternative in $W' \setminus W$, while ensuring the overall score is at least $\min\{\text{SC}_\lambda(W), \text{SC}_\lambda(W')\}$. Since $\text{SC}_\lambda(W) \leq \text{SC}_\lambda(W') + \delta_s$ per definition, we obtain a valid reconfiguration path in polynomial time. For $\delta_c = 1$, such a path has a shortest possible length $|W \setminus W'|$. For $\delta_c > 1$, we can take the same path, but skip $\delta_c - 1$ many committees in each step. This leads to a path of length $\lceil \frac{|W \setminus W'|}{\delta_c} \rceil$. Note that a shorter path cannot exist. $\quad\square$

**Example 3.** *We illustrate the greedy approach behind the proof of Theorem 1 on the following example under* AV*: Let $C = [6]$ and $V = \{v_1, \ldots, v_{14}\}$ with approval preferences $A_1 = \cdots = A_5 = \{1, 2, 3, 4\}$, $A_6 = A_7 = A_8 = \{3, 4, 6\}$, $A_9 = A_{10} = \{2, 5\}$, and $A_{11} = \cdots = A_{14} = \{1, 5, 6\}$. We have the following scores for each of the alternatives: $\text{SC}(1) = 9$, $\text{SC}(2) = 7$, $\text{SC}(3) = \text{SC}(4) = 8$, $\text{SC}(5) = 6$ and $\text{SC}(6) = 7$. Let k $= 3$, $\delta_c = 1$, $\delta_s = 5$, $W = \{1, 3, 4\}$, and $W' = \{2, 5, 6\}$. Note that $W$ is a winning committee with $\text{SC}(W) = 25$ and $\text{SC}(W') = 20$.*

*We now proceed with generating the reconfiguration path. We start with $W_0 = W$. Alternative 6 is the highest scoring alternative in $W' \setminus W_0$, while alternative 4 (or 3) the lowest scoring alternative in $W_0 \setminus W'$. Therefore, we generate $W_1 = \{1, 3, 6\}$ by exchanging 4 with 6. Using the same procedure, we compute $W_2 = \{1, 2, 6\}$, and finally $W_3 = \{2, 5, 6\} = W'$. It is straightforward to check that $(W_0 = W, W_1, W_2, W_3 = W')$ is a $(1, 5)$-reconfiguration path.*

In contrast to AV and SAV, the next theorem shows that CC- and PAV-MR are computationally difficult. The idea is to reduce from the PSPACE-complete and W[1]-hard problem INDEPENDENT SET RECONFIGURATION VIA TOKEN JUMPING (ISR-TJ) (Ito *et al.*, 2014), by adapting from the reduction of Aziz *et al.* (2015) for CC and PAV.

**Theorem 2** ($\star$). *For all* $\lambda \in \{CC, PAV\}$*,* $\lambda$*-MR$^0$ (and hence* $\lambda$*-MR) are PSPACE-complete. Moreover, they are PSPACE-hard and W[1]-hard wrt.* $k + \ell$*, even if* $(\delta_c, \delta_s, b) = (1, 0, 2)$*.*

We observe that the reduction of Ito *et al.* (2011) can be used to show that a generalized variant of ISR-TJ where in each step more than one vertex can be exchanged is NP-hard for constant path length. Hence, we obtain the following hardness for our problem as well.

**Proposition 2** ($\star$). *For all* $\lambda \in \{CC, PAV\}$*,* $\lambda$*-MR remains NP-hard even if* $b = \ell = 2$*, and for constant* $\ell$*, it is in NP.*

**Parameterized Complexity.** Next, we focus on the parameterized complexity for CC and PAV wrt. a few selected parameters. The most intuitive ones are the number $m$ of alternatives, the number $n$ of voters, and the committee size $k$.

**Proposition 3.** *For all score-based multi-winner voting rules* $\lambda$ *such that the score of a committee can be computed in polynomial time,* $\lambda$*-MR is solvable in* $m^{2k} \cdot (n + m)^{O(1)}$ *time, and hence FPT wrt.* $m$ *and in XP wrt.* $k$*. Furthermore a shortest path can be computed in the same time.*

*Proof.* Let $I = (\mathcal{P}, k, \delta_c, \delta_s, W, W')$ be an instance of $\lambda$-MR. We operate on the reconfiguration graph $G^R$ of $I$. As noted in Remark 3, the number of edges in $G^R$ is bounded above by $\binom{m}{k}^2 \leq m^{2k} \leq m^{2m}$, and hence $G^R$ can be constructed in $\binom{m}{k}^2 \cdot (n + m)^{O(1)}$ time. Since determining a shortest path between two vertices can be done in time linear in the number of edges of the graph, we can determine a shortest-length $(u_W, u_{W'})$-path in $G^R$ in $m^{2k} \cdot (n + m)^{O(1)}$ time, which is FPT wrt. $m$ and in XP wrt. $k$, as desired. $\square$

Yang and Wang (2023) show that PAV-MULTI-WINNER SCORE can be solved in FPT time wrt. $n$ via a specific integer linear programming (ILP) formulation called Integer Programming With Simple Piecewise Linear Transformation (IPWSPLT) that can encode convex constraints.[1] It is not clear how to extend their approach to the reconfiguration variant because the length $\ell$ of a shortest reconfiguration path may be unbounded. However, we can extend their approach to obtain an FPT result wrt. $n + \ell$.

**Theorem 3** ($\star$). *PAV-MR is FPT wrt.* $n + \ell$*.*

*Proof sketch.* As said, we extend the IPWSPLT by Yang and Wang (2023). Briefly put, IPWSPLT is an ILP extension that allows for piecewise concave or convex functions to be used (Bredereck *et al.*, 2020b). We construct $\ell$ many copies of the IPWSPLT formulation and combine them with additional constraints to model the exchanges. Similarly to their approach, we utilize the fact that alternatives of the same type are interchangeable when considering the score of a committee. We then split the alternatives of each type into 4 *groups*: Alternatives that are in $W \cap W'$, $W \setminus W'$, $W' \setminus W$, and $C \setminus (W \cup W')$. We add variables that keep track of the number of alternatives of each group from each type. By keeping track of these numbers along the steps of the reconfiguration path we ensure that the symmetric difference between two intermediate committees is at most $2\delta_c$ and that the final committee in the reconfiguration path is the target committee. We ensure that the total number of variables and constraints are in $O(2^n \cdot \ell)$. This yields an FPT algorithm wrt. $n + \ell$ since an IPWSPLT instance can be solved in time $O(\hat{n}^{2.5\hat{n}+o(\hat{n})} \cdot (|I| + p_{\max}))$ (Bredereck *et al.*, 2020b), where $\hat{n}$ is the number of integer variables, $|I|$ is the size of the encoding of the instance, and $p_{\max}$ is the maximum number of pieces of a piecewise-linear convex/concave function. Since in our IPWSPLT formulation the number $\hat{n}$ of variables and the size of the encoding $|I|$ are in $O(2^n \cdot \ell)$, the FPT result follows. The precise formulation and the proof are deferred to Appendix B.3. $\square$

The next results are based on either brute-force searching all possible reconfiguration paths or bounding the size of the reconfiguration graph.

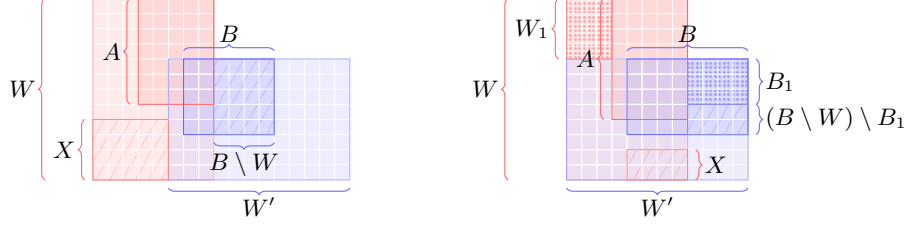

Figure 2: Illustration for the proof of Lemma 1. Left: Case 1; Right: Case 2.

**Proposition 4** ($\star$). *For all score-based voting rules $\lambda$ such that the score of a committee can be computed in polynomial time, the following holds:*

(i) *$\lambda$-MR is solvable in $m^{2\,\delta_{\mathrm{c}}\cdot\ell}\cdot(n+m)^{O(1)}$ time, and hence in XP wrt. $\ell$ for constant $\delta_{\mathrm{c}}$.*

(ii) *If $\lambda$ is additionally neutral, then $\lambda$-MR is solvable in $\mathsf{k}^{2\mathsf{k}}\cdot(2^n+2)^{2\mathsf{k}}\cdot(n+m)^{O(1)}$ time and $(\mathsf{k}+1)^{4\cdot2^n}(n+m)^{O(1)}$ time and a shortest reconfiguration path can be found in the same time. Thus, it is FPT wrt. $n+\mathsf{k}$ and in XP wrt. $n$.*

For CC, the following lemma shows that for a large enough committee size $\mathsf{k}$ there is always a reconfiguration path: The score of each committee is determined by at most $n$ relevant alternatives since each voter contributes a score of at most one. Due to this, we can find a shortest path if the symmetric difference between $W$ and $W'$ is very large (Case (i)), by only exchanging the alternatives in the symmetric difference. Otherwise (Case (ii)), we can still find a path in polynomial time by also exchanging alternatives that are shared by both $W$ and $W'$.

**Lemma 1.** *For every instance of* CC-MR *with $\mathsf{k} \geq 2n$ there exists a reconfiguration path between $W$ and $W'$. Moreover,*

(i) *If $|W \setminus W'| \geq 2n$, then we can find a shortest one in polynomial time.*

(ii) *If $|W \setminus W'| < 2n$, then we can find one with length at most $\lceil \frac{2n}{\delta_{\mathrm{c}}} \rceil$ in polynomial time.*

*Proof.* Let $I = (\mathcal{P}, \mathsf{k}, \delta_{\mathrm{c}}, \delta_{\mathrm{s}}, W, W')$ denote an instance of CC-MR. We first consider the case when $\delta_{\mathrm{c}} = 1$ and show how to adapt the proof for $\delta_{\mathrm{c}} \geq 2$ at the end. First, observe that we can always find a size-$n$ committee of $W$ (resp. $W'$) with score at least $\mathrm{SC}(W)$ (resp. $\mathrm{SC}(W')$). Hence, the idea is to find a size-$n$ committee $A \subseteq W$ (resp. $B \subseteq W'$) such that $\mathrm{SC}(A) = \mathrm{SC}(W)$ (resp. $\mathrm{SC}(B) = \mathrm{SC}(W')$) and perform exchanges to obtain a committee that contains $A \cup B$, and then switch out alternatives that are not contained in $W'$. Note that $A$ and $B$ can be found in polynomial time by iteratively deleting "redundant" alternatives from $W$ and $W'$, respectively. To show the statement, we distinguish between two cases. Figure 2 illustrates the constructions used in the proofs.

**Case 1:** $|W \setminus W'| \geq 2n$ or $|W \setminus (W' \cup A)| \geq |B \setminus W|$. Note that the first condition implies the second condition since $|A| = n = |B|$. To implement the exchange idea, let $X \subseteq W \setminus (W' \cup A)$ be an arbitrary committee of $|B \setminus W|$ alternatives. We divide the exchanges into two parts. In the first part, we step-by-step exchange all alternatives from $X$ with all alternatives from $B \setminus W$. It is straightforward to verify that every committee obtained until now has score at least $\mathrm{SC}(W)$ because it always contains $A$, and the last committee obtained has score at least $\mathrm{SC}(W')$ since it contains $B$. In the second part, we step-by-step exchange the remaining alternatives from $W \setminus (W' \cup X)$ with the remaining alternatives from $W' \setminus (W \cup B)$. It is straightforward to verify that every committee obtained in the second part has score at least $\mathrm{SC}(W') \geq \mathrm{SC}(W) - \delta_{\mathrm{s}}$ since it contains $B$. The exchange sequence yields a reconfiguration path of shortest length $\lceil |W \triangle W'|/2 \rceil$. If $\delta_{\mathrm{c}} \geq 2$, then we take every $\delta_{\mathrm{c}}$-th element on the path, including $W$ and $W'$. All these committees have score at least $\mathrm{SC}(W) - \delta_{\mathrm{s}}$ by construction and the path is of the shortest possible length $\lceil |W \triangle W'|/(2\,\delta_{\mathrm{c}}) \rceil$.

**Case 2:** $|W \setminus W'| < 2n$ and $|W \setminus (W' \cup A)| < |B \setminus W|$. We proceed in three parts. For the sake of brevity, let $W_1 = W \setminus (W' \cup A)$ and let $B_1 \subseteq B \setminus W$ denote a committee of $|W_1|$ alternatives arbitrarily chosen from $B \setminus W$. First, we step-by-step exchange the alternatives in $W_1$ with the alternatives in $B_1$. Afterwards, we select a committee $X \subseteq (W \cap W') \setminus (A \cup B)$ of $|B \setminus W| - |W_1|$ arbitrary alternatives, and step-by-step exchange the alternatives in $X$ with the remaining alternatives in $(B \setminus W) \setminus B_1$. Note that this is possible since $|(W \cap W') \setminus (A \cup B)| \geq |W \cap W'| - |A| + |A \setminus W'| - |B| + |B \setminus W| = |W| - |W \setminus W'| - |A| + |A \setminus W'| - |B| + |B \setminus W| \geq -|W_1| + |B \setminus W|$; the last inequality holds since

$|W_1| = |W \setminus (W' \cup A)| = |W \setminus W'| - |A \setminus W'|$ and $|W| \geq 2n \geq |A| + |B|$. So far, we perform $|B \setminus W| \leq n$ exchanges, where all intermediate committees contain $A$ and the final committee additionally contains $B$. Thus, they all have score at least $\text{SC}(W)$. The only alternatives from $W \setminus W'$ that remain in the most recently obtained committee are those alternatives in $A \setminus W'$. Thus, in the third part, we step-by-step exchange all alternatives in $A \setminus W'$ with the alternatives in $W'$ that are not yet in the most recent committee; they are precisely in $X \cup (W' \setminus (W \cup B))$. Since all committees obtained in the third part contain $B$, each of them has score $\text{SC}(W') \geq \text{SC}(W) - \delta_{\text{s}}$. The number of exchanges in the last part is $|A \setminus W'| \leq n$, and overall we have performed at most $2n$ exchanges. Similarly to the first case, if $\delta_{\text{c}} \geq 2$, then the number of exchanges reduces to $\lceil 2n/\delta_{\text{c}} \rceil$. $\qquad \square$

Combining Lemma 1 with Proposition 4(ii) yields an FPT result (wrt. $n$) for CC.

**Theorem 4** ($\star$). CC-MR *is FPT wrt.* $n$ *and a shortest reconfiguration path can be found in FPT time wrt.* $n$.

*Proof.* We distinguish between two cases. If $k < 2n$, then we use Proposition 4(ii) to solve CC-MR and find a shortest path in FPT-time wrt. $n$. Otherwise, by Lemma 1, we can in polynomial time find a reconfiguration path that either has minimum length (when $|W \setminus W'| \geq 2n$) or has length at most $\lceil \frac{2n}{\delta_{\text{c}}} \rceil$ (when $|W \setminus W'| < 2n$). This proves the first part of the statement.

To show the second part of the statement we only need to consider the case when $k \geq 2n$ and $|W \setminus W'| < 2n$ as the other cases are already covered by the above reasoning and by Lemma 1(i). In the case of $k \geq 2n$ and $|W \setminus W'| < 2n$, by Lemma 1(ii), there is always a reconfiguration path of length at most $\lceil \frac{2n}{\delta_{\text{c}}} \rceil$. This implies that a shortest path can exchange at most $4n$ many alternatives. Since CC is neutral, any two alternatives of the same type are "interchangeable" on the path. Hence, we only need to keep track of at most $4n$ alternatives for each type of alternative. Note that all alternatives in the symmetric difference $W \Delta W'$ must be among these tracked alternatives. There are in total at most $4n \cdot 2^n$ many alternatives that are relevant for the exchanges. Thus, we can use a simple branching algorithm to search for a shortest reconfiguration path by branching over all possibilities of selecting $2\delta_{\text{s}}$ alternatives from at most $4n2^n$ relevant ones to exchange in each step.

It remains to analyze the running time. In the case where $k < 2n$, the time needed to compute a shortest path is still $2n^{4n} \cdot (2^n + 2)^{4n} \cdot (n + m)^{O(1)}$ per Proposition 4(ii). For $k \geq 2n$ and $|W \setminus W'| \geq 2n$ a shortest path can be found in polynomial time per Lemma 1(i).

We now consider the case of $k \geq 2n$ and $|W \setminus W'| < 2n$. Before looking at the running time, we describe the branching algorithm more precisely. We note that we can assume $\delta_{\text{c}} < 2n$, as otherwise there is a path of length one that is obviously a shortest path. Recall that due to CC being neutral, it suffices to keep track of $4n$ alternatives of each type since by Lemma 1(ii) the maximum number of alternatives that are exchanged on a shortest path is $4n$. Let $X$ denote a set consisting of $\min(4n, |\mathsf{C}_t|)$ alternatives of each type $t \in \mathcal{T}$, including all alternatives from $W \Delta W'$. This implies that $|X| \leq 4n2^n$. As we know that the reconfiguration path has length at most $\ell \leq \lceil 2n/\delta_{\text{c}} \rceil$, we can solve this problem by branching over all possible exchanges using alternatives in $X$ with depth $\ell$. For each step $i \in [\ell]$, we branch over all pairs of two sets $X_i^1, X_i^2 \subseteq X$ with $|X_i^1| \leq \delta_{\text{c}}$ and $|X_i^2| \leq \delta_{\text{c}}$ that shall correspond to the set of alternatives in $W_{i-1} \setminus W_i$ and $W_i \setminus W_{i-1}$, respectively. Then, using these exchanges, we can compute all committees on each branch and check whether they yield a desired reconfiguration path. By checking all branches we can then find a shortest path.

Finally, we compute the running time of the branching algorithm. A single step has $|X|^{\delta_{\text{c}}} \cdot |X|^{\delta_{\text{c}}} = |X|^{2\delta_{\text{c}}}$ many branches and there are at most $\lceil \frac{2n}{\delta_{\text{c}}} \rceil$ steps. That means the branching algorithm has a running time of $|X|^{2\delta_{\text{c}}\ell} \leq |X|^{2\delta_{\text{c}}\lceil \frac{2n}{\delta_{\text{c}} \rceil}} \leq |X|^{4n+2\delta_{\text{c}}} \leq (4n \cdot 2^n)^{8n}$. $\qquad \square$

By observing that there are not too many different alternatives (wrt. $n + \mathsf{b}$), we obtain the following FPT result.

**Proposition 5** ($\star$). *For all neutral and score-based multi-winner voting rules $\lambda$ such that the score of a committee can be computed in polynomial time, $\lambda$-MR is solvable in $2^{2\mathsf{b}n}(n+m)^{O(1)}$ time, and hence FPT wrt. $n + \mathsf{b}$.*

Finally, we prove that the committee size $k$ combined with the path length $\ell$ is unlikely to yield FPT-algorithms by showing W[2]-hardness. We reduce from the problem of DOMINATING SET

RECONFIGURATION VIA TOKEN JUMPING (DSR-TJ). In DSR-TJ, we are given a graph $G = (V, E)$, two non-negative integers $h$ and $\hat{\ell}$, and two size-$h$ dominating sets $V_s$ and $V_t$ of $G$, and we ask whether there exists a sequence $V_0, \dots, V_{\hat{\ell}}$ of size-$h$ dominating set such that $V_0 = V_s$, $V_{\hat{\ell}} = V_t$ and $|V_i \Delta V_{i+1}| \leq 2$ for all $0 \leq i \leq \hat{\ell} - 1$. Recall that a *dominating set* of a graph is a subset $V'$ of vertices whose neighborhood covers the whole vertex set of the graph, i.e., each vertex not from $V'$ is adjacent to at least one vertex in $V'$. Bodlaender *et al.* (2021) claimed that Mouawad *et al.* (2016) showed that DSR-TJ is W[2]-hard wrt. $(h, \hat{\ell})$, where $h$ denotes the size of any dominating set (DS for short) on the path and $\hat{\ell}$ the length of a shortest DS reconfiguration path (if it exists). The reduction given by Mouawad et al., however, only works for the TAR but not the TJ variant. The reason is that in their created instance, the size $h'$ of the initial DS is smaller than $h$, so the size of any intermediate dominating set could be larger than $h'$ while this is not possible for the TJ variant. Below, we modify their reduction and show hardness for the TJ variant.

**Proposition 6 ($\star$).** DSR-TJ *is W[2]-hard wrt.* $h + \hat{\ell}$.

We obtain the same hardness for the CC rule by adapting the reduction by Betzler *et al.* (2013).

**Theorem 5.** CC-MR$^0$ *(and hence* CC-MR*) are W[2]-hard wrt.* $\mathsf{k} + \ell$.

*Proof.* It suffices to show W[2]-hardness for CC-MR$^0$. To achieve this, we provide a parameterized reduction from the W[2]-hard DSR-TJ problem (see Proposition 6) to CC-MR$^0$, i.e., CC-MR where $\delta_s = 0$ and $\delta_c = 1$.

Let $(G, h, \hat{\ell}, V_s, V_t)$ be an instance of DSR-TJ, where $G$ is an undirected graph, $V_s$ and $V_t$ are the initial and target dominating sets, respectively, $h = |V_s| = |V_t|$, and $\hat{\ell}$ denotes the length bound on the reconfiguration path (if it exists).

We define $\mathsf{k} = h$ and create a preference profile as follows. For each vertex $v_i \in V(G)$, create

– one *vertex-alternative* $c_i$, and
– one *vertex-voter* $u_i$ that approves of $c_i$ and the alternatives corresponding to the neighbors of $v_i$ in $G$. Formally, the approval set of voter $u_i$ is $\mathsf{A}_i = \{c_i\} \cup \{c_j \mid \{v_i, v_j\} \in E(G)\}$.

Let $\mathcal{P} = (C, V, R)$ denote the created profile. The total number of voters in this instance is $|V(G)|$. To complete the construction, let the initial and target committees be $W = \{c_i \mid v_i \in V_s\}$ and $W' = \{c_i \mid v_i \in V_t\}$. Note that the score of $W$ (resp. $W'$) is $|V(G)|$ since $V_s$ (resp. $V_t$) is a dominating set. It is straightforward to see that every size-$h$ dominating set of $G$ corresponds to a size-$\mathsf{k}$ committee with score at least $|V(G)|$. The correctness of the construction follows immediately. $\square$

## 4 Conclusion

We introduced a framework for reconfiguring committees in multi-winner elections for four prominent committee election rules: CC, PAV, AV, and SAV, and systematically studied the (parameterized) complexity of the problem. We left open whether our reconfiguration problem under PAV is FPT wrt. the number $n$ of voters.

Our framework can be directly applied to other score-based rules, such as the Monroe rule (Monroe, 1995), even for linear preferences. For non-score-based rules, we may adapt by either restricting the reconfiguration path to winning committees or introducing a measure that captures closeness to optimality.

Our preliminary experimental investigations (see Appendix C) indicate that for most randomly generated committees, a reconfiguration path not only exists but can also be efficiently determined using a straightforward heuristic. This finding opens a promising avenue for future research: determining the specific structures within the data that ensure the existence of a reconfiguration path, and whether it is possible to develop FPT algorithms leveraging these structures.

The concept of reconfiguration extends beyond the multi-winner election settings explored here or the fair division setting explored by Igarashi *et al.* (2024). Many other social choice frameworks remain unexplored in this context, such as stable matching (Gale and Shapley, 1962) or coalition formation (Drèze and Greenberg, 1980). Investigating these could provide useful insights into the algorithmic principles of varied decision-making processes.

## Acknowledgments

The authors are supported by the Vienna Science and Technology Fund (WWTF) [10.47379/ VRG18012]. We would like to thank the reviewers for their helpful comments. We acknowledge the support of the NeurIPS 2024 Financial Assistance grant, which helped the authors attend the conference.

## Footnotes

[1] ILP with convex constraints was shown to be FPT wrt. $n$ before the introduction of IPWSPLT by Grötschel *et al.* (1988), but we use IPWSPLT to keep the terminology consistent with Yang and Wang.

## References

Ravindra K. Ahuja, Özlem Ergun, James B. Orlin, and Abraham P. Punnen. A survey of very large-scale neighborhood search techniques. *Discrete Applied Mathematics*, 123(1-3):75–102, 2002.

Oswin Aichholzer, Wolfgang Mulzer, and Alexander Pilz. Flip distance between triangulations of a simple polygon is np-complete. *Discrete and Computational Geometry*, 54(2):368–389, 2015.

Soroush Alamdari, Patrizio Angelini, Fidel Barrera-Cruz, Timothy M. Chan, Giordano Da Lozzo, Giuseppe Di Battista, Fabrizio Frati, Penny Haxell, Anna Lubiw, Maurizio Patrignani, Vincenzo Roselli, Sahil Singla, and Bryan T. Wilkinson. How to morph planar graph drawings. *SIAM Journal on Computing*, 46(2):824–852, 2017.

Haris Aziz, Serge Gaspers, Joachim Gudmundsson, Simon Mackenzie, Nicholas Mattei, and Toby Walsh. Computational aspects of multi-winner approval voting. In *Proceedings of the 14th International Conference on Autonomous Agents and Multiagent Systems, (AAMAS 2015)*, pages 107–115, 2015.

Nadja Betzler, Arkadii Slinko, and Johannes Uhlmann. On the computation of fully proportional representation. *Journal of Artificial Intelligence Research*, 47:475–519, 2013.

Hans L. Bodlaender, Carla Groenland, and Céline M. F. Swennenhuis. Parameterized complexities of dominating and independent set reconfiguration. In *Proceedings of the 16th International Symposium on Parameterized and Exact Computation, (IPEC 2021)*, pages 9:1–9:16, 2021.

Niclas Boehmer and Rolf Niedermeier. Broadening the research agenda for computational social choice: Multiple preference profiles and multiple solutions. In *Proceedings of the 20th International Conference on Autonomous Agents and Multiagent Systems, (AAMAS 2021)*, pages 1–5, 2021.

Paul Bonsma. The complexity of rerouting shortest paths. *Theoretical Computer Science*, 510:1–12, 2013.

Robert Bredereck, Jiehua Chen, Dušan Knop, Junjie Luo, and Rolf Niedermeier. Adapting stable matchings to evolving preferences. In *Proceedings of the 34th AAAI Conference on Artificial Intelligence (AAAI 2020)*, pages 1830–1837, 2020.

Robert Bredereck, Piotr Faliszewski, Rolf Niedermeier, Piotr Skowron, and Nimrod Talmon. Mixed integer programming with convex/concave constraints: Fixed-parameter tractability and applications to multicovering and voting. *Theoretical Computer Science*, 814:86–105, 2020.

Robert Bredereck, Andrzej Kaczmarczyk, and Rolf Niedermeier. Electing successive committees: Complexity and algorithms. In *Proceedings of the 34th AAAI Conference on Artificial Intelligence (AAAI 2020)*, pages 1846–1853, 2020.

Robert Connelly, Erik D. Demaine, and Günter Rote. Blowing up polygonal linkages. *Discrete & Computational Geometry*, 30(2):205–239, 2003.

Marek Cygan, Fedor V. Fomin, Lukasz Kowalik, Daniel Lokshtanov, Dániel Marx, Marcin Pilipczuk, Michal Pilipczuk, and Saket Saurabh. *Parameterized Algorithms*. Springer, 2015.

Rodney G. Downey and Michael R. Fellows. *Fundamentals of Parameterized Complexity*. Springer, 2013.

J. H. Drèze and J. Greenberg. Hedonic coalitions: Optimality and stability. *Econometrica*, 48(4):987–1003, 1980.

Piotr Faliszewski, Piotr Skowron, Arkadii Slinko, and Nimrod Talmon. Multiwinner voting: A new challenge for social choice theory. *Trends in computational social choice*, 74:27–47, 2017.

David Gale and Lloyd S. Shapley. College admissions and the stability of marriage. *The American Mathematical Monthly*, 120(5):386–391, 1962.

Sevag Gharibian and Jamie Sikora. Ground state connectivity of local hamiltonians. *ACM Transaction on Computation Theory*, 10(2):8:1–8:28, 2018.

Parikshit Gopalan, Phokion G. Kolaitis, Elitza N. Maneva, and Christos H. Papadimitriou. The connectivity of boolean satisfiability: Computational and structural dichotomies. In *Proceedings of the 33rd International Colloquium on Automata, Languages and Programming (ICALP 2006)*, pages 346–357, 2006.

Martin Grötschel, László Lovász, and Alexander Schrijver. *Geometric Algorithms and Combinatorial Optimization*, volume 2 of *Algorithms and Combinatorics*. Springer, 1988.

Ayumi Igarashi, Naoyuki Kamiyama, Warut Suksompong, and Sheung Man Yuen. Reachability of fair allocations via sequential exchanges. In *Proceedings of the 38th AAAI Conference on Artificial Intelligence (AAAI 2024)*, pages 9773–9780, 2024.

Takehiro Ito, Erik D. Demaine, Nicholas J. A. Harvey, Christos H. Papadimitriou, Martha Sideri, Ryuhei Uehara, and Yushi Uno. On the complexity of reconfiguration problems. *Theoretical Computer Science*, 412(12-14):1054–1065, 2011.

Takehiro Ito, Marcin Kaminski, Hirotaka Ono, Akira Suzuki, Ryuhei Uehara, and Katsuhisa Yamanaka. On the parameterized complexity for token jumping on graphs. In *11th International Conference on Theory and Applications of Models of Computation (TAMC 2022)*, pages 341–351, 2014.

Takehiro Ito, Yuni Iwamasa, Naonori Kakimura, Naoyuki Kamiyama, Yusuke Kobayashi, Yuta Nozaki, Yoshio Okamoto, and Kenta Ozeki. Reforming an envy-free matching. In *Proceedings of the 36th AAAI Conference on Artificial Intelligence (AAAI 2022)*, pages 5084–5091, 2022.

William W. Johnson and William E. Story. Notes on the "15" puzzle. *American Journal of Mathematics*, 2(4):397–404, 1879.

Chengquan Ju, Shuhan Yao, and Peng Wang. Resilient post-disaster system reconfiguration for multiple energy service restoration. In *Proceedings of the 1st IEEE Conference on Energy Internet and Energy System Integration ($EI^2$ 2017)*, pages 1–6, 2017.

Martin Lackner and Piotr Skowron. *Multi-Winner Voting with Approval Preferences–Artificial Intelligence, Multiagent Systems, and Cognitive Robotics*. Springer Briefs in Intelligent Systems. Springer, 2023.

Martin Lackner, Peter Regner, and Benjamin Krenn. abcvoting: A Python package for approval-based multi-winner voting rules. *Journal of Open Source Software*, 8(81):4880, 2023.

Burt L Monroe. Fully proportional representation. *American Political Science Review*, 89(4):925–940, 1995.

Amer E. Mouawad, Naomi Nishimura, Venkatesh Raman, Narges Simjour, and Akira Suzuki. On the parameterized complexity of reconfiguration problems. *Algorithmica*, 78(1):274–297, may 2016.

Netflix. Netflix prize. `https://www.kaggle.com/datasets/netflix-inc/netflix-prize-data`, 2006. Accessed 2023-7-7, Version 2.

Netflix. Don't Miss Your Chance to Watch These Movies and Shows Leaving in December. `https://www.netflix.com/tudum/articles/whats-leaving-netflix`, 2024. [Accessed 17-01-2024].

Naomi Nishimura. Introduction to reconfiguration. *Algorithms*, 11(4):52, 2018.

Prime-Video. Prime video: leaving soon. `https://www.amazon.co.uk/Expiring-Prime-Next-30-Days-Videos/s?rh=n%3A9791716031%2Cp_n_ways_to_watch%3A7448660031`, 2024. [Accessed 17-01-2024].

Walter J. Savitch. Relationships between nondeterministic and deterministic tape complexities. *Journal of Computer System Science*, 4(2):177–192, 1970.

Baruch Schieber, Hadas Shachnai, Gal Tamir, and Tami Tamir. A theory and algorithms for combinatorial reoptimization. *Algorithmica*, 80(2):576–607, 2018.

Stanisław Szufa, Piotr Faliszewski, Łukasz Janeczko, Martin Lackner, Arkadii Slinko, Krzysztof Sornat, and Nimrod Talmon. How to sample approval elections? In *Proceedings of the 31st International Joint Conference on Artificial Intelligence (IJCAI 2022)*, pages 496–502, 2022.

Jan van den Heuvel. The complexity of change. In Simon R. Blackburn, Stefanie Gerke, and Mark Wildon, editors, *Surveys in Combinatorics 2013*, volume 409 of *London Mathematical Society Lecture Note Series*, pages 127–160. Cambridge University Press, 2013.

Yongjie Yang and Jianxin Wang. Parameterized complexity of multiwinner determination: more effort towards fixed-parameter tractability. *Autonomous Agents and Multi-Agent Systems*, 37(2):28, 2023.

# Supplementary Material for the Paper "Multi-Winner Reconfiguration"

## A   Additional Material for Section 2

**Parameters.**  In this work we additionally look at the problems through the lens of parameterized complexity using the following parameters: the number $m$ of alternatives, the number $n$ of voters, the committee size $k$, the maximum number $b$ of approved alternatives per voter and the length $\ell$ of a shortest reconfiguration path (if it exists).

## B   Additional Material for Section 3

### Independent Set Reconfiguration Definition

Ito *et al.* (2014) show that the following reconfiguration is W[1]-hard wrt. the number of tokens (i.e., the size of any independent set on the path).

INDEPENDENT SET RECONFIGURATION VIA TOKEN JUMPING (ISR-TJ)
**Input:** A graph $G$, two non-negative integers $h$ and $\hat{\ell}$, and two independent sets $I_s, I_t$ of size $h$.
**Question:** Does there exist a sequence $I_0, \ldots, I_{\hat{\ell}}$ of size-$h$ independent sets of $G$ such that
$I_0 = I_s$, $I_{\hat{\ell}} = I_t$, and $|I_i \Delta I_{i+1}| \leq 2$ holds for all $0 \leq i \leq \hat{\ell} - 1$?

An *independent set* of a graph is a subset of vertices such that no two of them are adjacent.

### B.1   Proof of Theorem 2

**Theorem 2** ($\star$).  *For all $\lambda \in \{CC, PAV\}$, $\lambda$-MR$^0$ (and hence $\lambda$-MR) are PSPACE-complete. Moreover, they are PSPACE-hard and W[1]-hard wrt. $k + \ell$, even if $(\delta_c, \delta_s, b) = (1, 0, 2)$.*

*Proof.*  Since PSPACE-membership follows from Proposition 1, we only need to focus on the hardness results. We show W[1]-hardness and PSPACE-hardness by reducing from ISR-TJ (see above for the definition). Ito *et al.* (2014) show that ISR-TJ is both PSPACE-complete and W[1]-hard wrt. to $h$. We observe that the reduction is simultaneously a polynomial and W[1]-hardness reduction wrt. $h + \hat{\ell}$ because a shortest reconfiguration path has length $2h$ in their construction. Hence, if we can provide a polynomial-time reduction for CC-MR$^0$ and PAV-MR$^0$ such that the committee size and the path length are $h$ and $\hat{\ell}$, respectively, then we obtain W[1]-hardness in the statement.

Let $(G, h, I_s, I_t)$ be an instance of ISR-TJ. We create an instance of CC-MR$^0$ and PAV-MR$^0$ with committee size $k = h$. For each vertex $v_i \in V(G)$, let $\deg(v_i)$ denote the degree of vertex $v_i$ in $G$. Further let $\deg(G) := \max_{v_i \in V(G)} \deg(v_i)$ denote the maximum degree of graph $G$.

- For each vertex $v_i \in V(G)$, we create a *vertex-alternative* $c_i$ and $(\deg(G) - \deg(v_i))$-many private dummy voters $\{d_i^1, \ldots, d_i^{\deg(G) - \deg(v_i)}\}$ that only approve $c_i$.
- For each edge $\{v_i, v_j\} \in E(G)$, we create one *edge-voter* $u_{\{i,j\}}$ that approves only $c_i$ and $c_j$.

Let $\mathcal{P}$ denote the created profile with $C = \{c_i \mid v_i \in V(G)\}$ and $V = \{d_i^z \mid v_i \in V(G), z \in [\deg(G) - \deg(v_i)]\} \cup \{u_{i,j} \mid \{v_i, v_j\} \in E(G)\}$. The total number of voters created is $|V| = \deg(G) \cdot |V(G)|$. To complete the construction, let the initial and target committees be $W = \{c_i \mid v_i \in I_s\}$ and $W' = \{c_i \mid v_i \in I_t\}$, the size of the independent set $h = k$, and the length of a reconfiguration path $\ell = 2h = 2k$.

For the sake of brevity, define a *bijection* $f \colon 2^{V(G)} \to 2^C$ with $f(V') = \{c_i \mid v_i \in V'\}$ between the vertex subsets and the committees. We present a one-to-one correspondence between the size-$h$ independent sets in $G$ and size-$h$ committees with score of at least $\deg(G) \cdot k$ in the created PAV-MR$^0$- and CC-MR$^0$-instance.

**Claim 5.1.**  *$V'$ is a size-$h$ independent set if and only if $f(V')$ is a size-$h$ committee of score at least $\deg(G) \cdot k$ under CC as well as PAV.*

*Proof.* The "only if" part is straightforward: For each size-$h$ independent set $V'$ we have that the alternatives in $f(V')$ are approved by $\deg(G) \cdot \mathsf{k}$ voters. Among them $\deg(G) \cdot \mathsf{k} - \sum_{v_i \in f(V')} \deg(v_i)$ voters are private voters and $\sum_{v_i \in f(V')} \deg(v_i)$ voters are the edge-voters of the edges incident to $V'$. Each of these $\deg(G) \cdot \mathsf{k}$ edge-voters approves exactly one alternative in $f(V')$ since $V'$ is an independent set. Thus $\mathrm{sc}_{\mathrm{PAV}}(W) = \mathrm{sc}_{\mathrm{CC}}(W) = \deg(G) \cdot \mathsf{k}$.

For the "if" part, let $W''$ denote a size-$h$ committee with score $\geq \deg(G) \cdot \mathsf{k}$. We claim that $f^{-1}(W'')$ is a size-$h$ independent set. Clearly $|f^{-1}(W'')| = h$, so we only need to show that no edge is incident to more than one vertex in $f^{-1}(W'')$. Let $E_1 = \{\{v_i, v_j\} \in E(G) : |\{v_i, v_j\} \cap f^{-1}(W'')| = 1\}$ and $E_2 = \{\{v_i, v_j\} \in E(G) : |\{v_i, v_j\} \cap f^{-1}(W'')| = 2\}$ denote the set of edges that are incident to exactly one vertex in $f^{-1}(W'')$ and the set of edges that are incident to two vertices in $f^{-1}(W'')$, respectively. Then, by the handshaking lemma, we have that

$$|E_1| + 2|E_2| = \sum_{c_i \in W''} \deg(v_i). \tag{1}$$

For CC, we have that $\deg(G) \cdot \mathsf{k} = \mathrm{sc}_{\mathrm{CC}}(W'') = \sum_{c_i \in W''} \big( \deg(G) - \deg(v_i) \big) + |E_1| + |E_2|$, which implies that $|E_1| + |E_2| = \sum_{c_i \in W''} \deg(v)$ since $k = h$. Together with (1), we obtain that $|E_2| = 0$, meaning that no edge is incident to more than one vertex in $f^{-1}(W'')$, as desired.

As for PAV, since $\deg(G) \cdot \mathsf{k} = \mathrm{sc}_{\mathrm{PAV}}(W'') = \sum_{c_i \in W''} \big( \deg(G) - \deg(v_i) \big) + |E_1| + (1 + 1/2)|E_2|$, we infer that $|E_1| + (1 + 1/2)|E_2| = \sum_{c_i \in W''} \deg(v)$. Similarly to the previous case, by (1), we obtain that $|E_2| = 0$, meaning that no edge is incident to more than one vertex in $f^{-1}(W'')$, as desired. (of Claim 5.1) $\diamond$

By Claim 5.1, it is straightforward to verify that a sequence $(I_s, \ldots, I_t)$ of vertex subsets is a size-$h$ independent set reconfiguration path if and only if $(W = f(I_s), \ldots, W' = f(I_t))$ is a $(1, 0)$-reconfiguration for CC-MR and PAV-MR. This concludes the proof for the PSPACE-hardness since the reduction is a polynomial time reduction.

Note that in our construction each voter approved of at most 2 alternatives. This means the PSPACE-hardness remains even if the ballot size is bounded by 2.

It remains to show the W[1]-hardness wrt. the committee size $k$ and the path length $\ell$. As the committee size $\mathsf{k}$ is equal to the size $h$ of the independent set, each committee on the reconfiguration path directly corresponds to an independent set on the reconfiguration path of the independent sets of size $h$ and the shortest path is of length at most $2h$ (Ito *et al.*, 2014), it follows that both PAV-MR$^0$ and CC-MR$^0$ are W[1]-hard wrt. the path length $\ell$ and the committee size $\mathsf{k}$. This concludes the proof. $\square$

### B.2 Proof of Proposition 2

**Proposition 2** ($\star$). *For all* $\lambda \in \{\mathrm{CC}, \mathrm{PAV}\}$, $\lambda$-MR *remains NP-hard even if* $\mathsf{b} = \ell = 2$, *and for constant* $\ell$, *it is in NP.*

*Proof.* As already mentioned, the reduction of Ito *et al.* (2011) can be used to show NP-hardness for the case when we may allow for more than one token to be moved at once. In other words, it is NP-hard to decide whether one can reconfigure from one independent set $I_s$ of size $h$ to another independent set $I_t$ of $h$ via an intermediate step. Hence, by using the same reduction as for Theorem 2, we obtain the result from the statement.

As for constant path length $\ell$, we can guess a sequence of constantly many committees which in total has polynomial size and check in polynomial time whether it is a valid reconfiguration path. This proves containment in NP. $\square$

### B.3 Continuation of the proof of Theorem 3

**Theorem 3** ($\star$). PAV-MR *is FPT wrt.* $n + \ell$.

We first introduce some notations and concepts that are useful for the IPWSPLT formulation. For each type $t$ of alternative, let $\mathsf{C}_t$ denote the set consisting of all alternatives with type $t$. Let $\mathcal{T}$ denote

the set of types of alternatives that exist in a given preference profile $\mathcal{P}$. Further, let $A(t)$ denote the set of voters that approve of alternatives of type $t$. We say that two committees $C_1$ and $C_2$ are in the same *equivalence class* (*class* for short) if for each type $t$, the number of alternatives with type $t$ in $C_1$ is the same as that in $C_2$. We partition the set $\mathsf{C}_t$ of alternatives of type $t$ into four subsets as follows: $X_t^1 = \mathsf{C}_t \cap (W \setminus W')$, $X_t^2 = \mathsf{C}_t \cap (W \cap W')$, $X_t^3 = \mathsf{C}_t \cap (W' \setminus W)$, and $X_t^4 = \mathsf{C}_t \cap (C \setminus (W' \cup W))$. In other words, $X_t^1$, $X_t^2$, $X_t^3$, and $X_t^4$ denote the set consisting of all alternatives of type $t$ that are from the set $W \setminus W'$, the intersection $W \cap W'$, the other set $W' \setminus W$, and not in $W \cup W'$, respectively.

We now describe the variables and constraints used in the IPWSPLT. We use IPWSPLT rather than a regular ILP because it is not clear how to compute the scores under PAV with just ILP.

**Variables.** The following variables will be used:

- An integer variable $x_{t,j}^1$ for each $t \in \mathcal{T}$ and each $j \in \{0, \ldots, \ell\}$. This variable stores the number of alternatives in $X_t^1$ in the $j$-th committee of the reconfiguration graph.
- An integer variable $x_{t,j}^2$ for each $t \in \mathcal{T}$ and each $j \in \{0, \ldots, \ell\}$. This variable stores the number of alternatives in $X_t^2$ in the $j$-th committee of the reconfiguration graph.
- An integer variable $x_{t,j}^3$ for each $t \in \mathcal{T}$ and each $j \in \{0, \ldots, \ell\}$. This variable stores the number of alternatives in $X_t^3$ in the $j$-th committee of the reconfiguration graph.
- An integer variable $x_{t,j}^4$ for each $t \in \mathcal{T}$ and each $j \in \{0, \ldots, \ell\}$. This variable stores the number of alternatives in $X_t^4$ in the $j$-th committee of the reconfiguration graph.
- An integer variable $y_{(t,j-1),(t',j)}^{z,z'}$ for each $z, z' \in [4]$, $j \in [\ell]$ and $t, t' \in \mathcal{T}$. This variable stores the number of alternatives that were part of the group represented by $x_{t,j-1}^z$ that were replaced by alternatives in the group $x_{t',j}^{z'}$ in the step from $W_{j-1}$ to $W_j$.
- An integer variable $x_{v,j}$ for each voter $v \in V$ and each step $j \in \{0, \ldots, \ell\}$. This variable stores how many alternatives voter $v$ approves of in committee $W_j$.

We use the same piecewise linear concave function $f \colon \mathbb{R}_{\geq 0} \to \mathbb{R}_{\geq 0}$ as Yang and Wang. We set $f(0) = 0$ and for every positive integer $x$ we set $f(x) = \sum_{i=1}^{x} \frac{1}{i}$. Then for each real positive non-integer $x$ we choose $y \in \mathbb{N}$ such that $y = \lfloor x \rfloor$ and set $f(x) = f(y) + (x - y) \cdot (f(y+1) - f(y))$.

**Constraints.** We add the following constraints:

(1) For each $z \in [4]$, we fix $x_{t,0}^z$ and $x_{t,\ell}^z$ to be the correct values for $W$ and $W'$, i.e., $x_{t,0}^1 = |X_t^1|$, $x_{t,0}^2 = x_{t,\ell}^2 = |X_t^2|$, $x_{t,0}^3 = x_{t,0}^4 = x_{t,\ell}^1 = x_{t,\ell}^4 = 0$, and $x_{t,\ell}^3 = |X_t^3|$.

(2) For each $j \in [\ell]$, we add constraint $\sum_{\substack{z,z' \in [4] \\ t,t' \in \mathcal{T}}} y_{(t,j-1),(t',j)}^{z,z'} \leq \delta_{\mathrm{c}}$. These constraints ensure that the symmetric difference between concurrent committees in the reconfiguration path observes the bound.

(3) For each $(j, t, z) \in \{0, \ldots, \ell\} \times \mathcal{T} \times [4]$, we add constraint $x_{t,j}^z \leq |X_t^z|$. These constraints ensure that the class of committee used in an intermediary result exists, i.e., we do not use more alternatives of a type in a committee than possible.

(4) For each $(j, t, z) \in [\ell] \times \mathcal{T} \times [4]$, we add a constraint $x_{t,j-1}^z \geq \sum_{\substack{t' \in \mathcal{T} \\ z' \in [4]}} y_{(t,j-1),(t',j)}^{z,z'}$. Theses constraints are added so that it is not possible to switch out alternatives that are not in the committee.

(5) For each $(j, t, z) \in [\ell] \times \mathcal{T} \times [4]$, we add a constraint $x_{t,j}^z = x_{t,j-1}^z + \sum_{z' \in [4], t' \in \mathcal{T}} y_{(t',j-1),(t',j)}^{z',z} - \sum_{z' \in [4], t' \in \mathcal{T}} y_{(t,j-1),(t',j)}^{z,z'}$. These constraints ensure that the number of alternatives in each group is as described after the exchanges.

(6) For each $j \in \{0, \ldots, \ell\}$, we add constraint $\sum_{t \in \mathcal{T}, z \in [4]} x_{t,j}^z = \mathsf{k}$. This constraint ensures that each committees of the reconfiguration path has size $\mathsf{k}$.

(7) For each $v \in V$ and $i \in [\ell]$ we add a constraint $x_{v,i} = \sum_{t \colon v \in A(t), z \in [4]} x_{t,j}^z$. These constraints link the number of alternatives of each type with the number of alternatives each voter approves of.

(8) For every $j \in [\ell]$ we add a constraint $\sum_{v \in V} f(x_{v,j}) \geq \mathrm{SC}_{\mathrm{PAV}}(\mathcal{P}, W) - \delta_{\mathrm{s}}$. These constraints ensure that each committee in the reconfiguration has a high enough score.

The number of variables and constraints are both $O(2^n \cdot \ell)$ and the function $f$ consists of at $\mathsf{k}$ linear parts and hence this IPWSPLT can be solved in FPT time wrt. $n + \ell$. Note that the constraints (3)-(8)

are the constraints used by Yang and Wang for the $\ell$ committees on the reconfiguration graph.

We now show that the constructed IPWSPLT feasibility instance is a yes-instance if and only if there exists a reconfiguration path of at most length $\ell$ between the given committees $W$ and $W'$.

The if part is straightforward: Let $W_0, \ldots, W_\ell$ be a $(\delta_c, \delta_s)$-reconfiguration path for $(\mathcal{P}, \lambda)$. For every $t \in \mathcal{T}, j \in \{0, \ldots, \ell\}$, we set the variable $x_{t,j}^z$ to be the number of alternatives in $W_i$ that are also in the set represented by $X_t^z$, i.e., $|W_i \cap X_t^z|$. We additionally set for every $z, z' \in [4]$ the variable $y_{(t,j-1),(t',j)}^{z,z'}$ to be the number of alternatives of type $t$ that were replaced by alternatives of type $t'$ between the committees $W_{j-1}$ and $W_j$. Constraint (5) is thus satisfied by construction. As these variables describe a valid committee, the constraints (6)-(8) are satisfied, as shown by Wang and Yang. Constraint (1) is satisfied, as $W_0 = W$ and $W_\ell = W'$. Constraint (2) is satisfied because the symmetric difference is upper bounded by $2\,\delta_c$. Constraint (3) must be satisfied, as it is not possible to have more alternatives of any type in the solution then there are in total. Similarly constraint (4) must be satisfied, as we can only switch out alternatives from groups that were previously in the solution.

The only-if part follows similarly: We iteratively create the committees $W_j$ for every $j \in [\ell]$. Note that $W_0 = W$ and $W_\ell = W'$ are already fixed. Suppose the committees $W_1, \ldots, W_j$ have been created and fulfill our conditions. We then create $W_{j+1}$. For each variable $y_{(t,j),(t',j+1)}^{z,z'}$, we replace that many alternatives that are in $W_j$ and belong to the group $|X_t^z|$ with alternatives that are not in $W_j$ and belong to $|X_{t'}^{z'}|$. After this process, we have replaced at most $\delta_c$ many alternatives with $\delta_c$ many other alternatives due to Constraint (2), and therefore the symmetric difference is upper bounded by $2\,\delta_c$. The resulting committee must exist because Constraint (3) ensures that each group in each type of alternatives does not appear more often in a committee then possible and Constraint (4) ensures that only existing alternatives were switched out. Constraint (6) ensures that the committee has size $\mathsf{k}$. Constraint (7) counts the number of alternatives voters approve and with Constraint (8) ensures that the score is sufficiently high. As this reasoning holds for every $j \in [\ell]$, the generated $W_0, \ldots, W_\ell$ is a $(\delta_c, \delta_s)$-reconfiguration path from $W$ to $W'$, as required.

Therefore it is possible to find a path between two committees using the IPWSPLT feasibility formulation.

### B.4   Proof of Proposition 4

**Proposition 4** ($\star$). *For all score-based voting rules $\lambda$ such that the score of a committee can be computed in polynomial time, the following holds:*

  *(i)  $\lambda$-MR is solvable in $m^{2\,\delta_c \cdot \ell} \cdot (n+m)^{O(1)}$ time, and hence in XP wrt. $\ell$ for constant $\delta_c$.*
 *(ii)  If $\lambda$ is additionally neutral, then $\lambda$-MR is solvable in $\mathsf{k}^{2\mathsf{k}} \cdot (2^n + 2)^{2\mathsf{k}} \cdot (n+m)^{O(1)}$ time and $(\mathsf{k}+1)^{4 \cdot 2^n} (n+m)^{O(1)}$ time and a shortest reconfiguration path can be found in the same time. Thus, it is FPT wrt. $n + \mathsf{k}$ and in XP wrt. $n$.*

*Proof.* For Statement (i), we observe that in each reconfiguration step we exchange at most $\delta_c$ alternatives with at most $\delta_c$ other alternatives. This gives us $\binom{m}{\delta_c}^2$ possible different exchanges. Since for a yes-instance we have at most $\ell$ steps in the reconfiguration path, we have $(\binom{m}{\delta_c}^2)^\ell = m^{2\,\delta_c \cdot \ell}$ possible different reconfiguration sequences. Since the score of a committee can be computed in polynomial time, we can check for every sequence whether it is a $(\delta_c, \delta_s)$-reconfiguration path from $W$ to $W'$ in $m^{2\,\delta_c \cdot \ell} \cdot (n+m)^{O(1)}$ time.

The proof idea for Statement (ii) is to group alternatives of the same type, keep only a bounded number of alternatives for each type and then generate a reconfiguration graph of bounded size. To this end, we enumerate the types as $1, \ldots, 2^n$. Let $\mathcal{T}$ denote the set of types of alternatives that exist in a given preference profile $\mathcal{P}$. We define $\mathsf{C}_t$ to be the set consisting of all alternatives with type $t$ for $t \in \mathcal{T}$. We will analyze the running times one by one: Generally, let $\mathcal{P} = (C, V, R)$ be a preference profile and $I = (\mathcal{P}, \mathsf{k}, \delta_c, \delta_s, W, W')$ an instance of $\lambda$-MR.

For the first part of Statement (ii), we observe that every committee on a reconfiguration path contains at most $\mathsf{k}$ alternatives of each type. Therefore, apart from the alternatives that appear in $W \cup W'$, we

may remove all but k many alternatives of each type from the instance. Let us call the set of kept alternatives $\hat{V}$. This means that we only need to keep at most $2k + k \cdot 2^n = k(2^n + 2)$ alternatives. We can construct a restricted reconfiguration graph of the remaining alternatives and determine a shortest path between $u_W$ and $u_{W'}$ in time $\left(\binom{k(2^n+2)}{k}\right)^2 \cdot (n+m)^{O(1)} = (k(2^n+2))^{2k} \cdot (n+m)^{O(1)}$.

A $u_W$-$u_{W'}$ path of length $\ell'$ in the restricted reconfiguration graph corresponds to a $(\delta_c, \delta_s)$-reconfiguration path of length $\ell'$, as the committees corresponding to the vertices on the path form a $(\delta_c, \delta_s)$-reconfiguration path in $I$. Next assume we have a $(\delta_c, \delta_s)$-reconfiguration path $(W_0, \ldots, W_{\ell'})$, where $W = W_0$ and $W' = W_{\ell'}$, of length $\ell'$ in $I$. We will show that there is a $u_W$-$u_{W'}$ path of length at most $\ell'$ in the restricted reconfiguration graph. Let us iteratively construct the path $u_{\hat{W}_1}, \ldots, u_{\hat{W}_{\ell'}}$. We show that in each step the following two conditions hold for $j \in \{0, \ldots, \ell'\}$:

(C1) $W_{\ell'} \cap (W \cup W') = \hat{W}_{\ell'} \cap (W \cup W')$, i.e., an alternative from $W \cup W'$ is in $W_{\ell'}$ if and only if it is in $\hat{W}_{\ell'}$, and

(C2) for every alternative type $t \in \mathcal{T}$, it holds that $|(W_j \cap C_t) \setminus (W \cup W')| = |(\hat{W}_j \cap C_t) \setminus (W \cup W')|$, i.e, for every type, the number of alternatives of that type is the same in both $W$ and $W'$, discounting the alternatives in the initial and goal committees.

From these two conditions it follows clearly that $\text{SC}(\hat{W}_j) = \text{SC}(W_j)$ for every $j \in \{0, \ldots, t\}$. Since all alternatives in $W \cup W'$ are retained, the first vertex in the path is $u_W$ and the last $u_{W'}$. Assume we have created vertices $u_{\hat{W}_0}, \ldots, u_{\hat{W}_j}$ so that $\hat{W}_0, \ldots, \hat{W}_j$ satisfy the conditions. We construct the committee $\hat{W}_{j+1}$ from $\hat{W}_j$ as follows:

(i) We remove the alternatives in $(W_j \setminus W_{j+1}) \cap (W \cup W')$.
(ii) We add the alternatives in $(W_{j+1} \setminus W_j) \cap (W \cup W')$.
(iii) For every type $t \in \mathcal{T}$ such that $d := |W_j \setminus (W \cup W')| - |W_{j+1} \setminus (W \cup W')| > 0$, we remove arbitrary $d$ alternatives of type $t$ from $\hat{W}_j \setminus (W \cup W')$.
(iv) For every type $t \in \mathcal{T}$ such that $d := |W_{j+1} \setminus (W \cup W')| - |W_j \setminus (W \cup W')| > 0$, we add arbitrary $d$ alternatives of type $t$ from $(\hat{V} \cap C_t \setminus \hat{W}_j) \setminus (W \cup W')$.

We show that these exchanges, if possible to perform, enforce that $\hat{W}_{j+1}$ satisfies the two conditions (C1) and (C2) we set to prove. The first two exchanges guarantee (C1), as they remove and add the corresponding alternatives to $\hat{W}_{j+1}$. The last two guarantee (C2), as they remove and add alternatives of the same type as are removed and from $W_j$ to $W_{j+1}$. It is obvious that Exchanges (i) and (ii) can be performed. Similarly, Exchange (iii) is clearly possible as we remove alternatives that are necessarily in the committee. To see that we can always perform Exchange (iv), recall that $\hat{V}$ contains $\hat{d}_t := \min(k, |C_t \cap C \setminus (W \cup W')|)$ alternatives of type $t$. No committee may contain more than $\hat{d}_t$ many alternatives of $C_t \setminus (W \cup W')$, so this exchange is also safe. It is also clear we will always add at most $|W_j \setminus W_{j+1}|$ alternatives and remove at most $|W_{j+1} \setminus W_j|$ alternatives, so $\hat{W}_j$ and $\hat{W}_{j+1}$ are $\delta_c$-adjacent, and thus $u_{\hat{W}_j}$ and $u_{\hat{W}_{j+1}}$ are adjacent. This concludes the proof of the first part of Statement (ii).

For the second part of Statement (ii), we observe that there can be at most k alternatives of each type in a committee, so we can upper-bound the number of committees we need to track to $(k+1)^{2 \cdot 2^n}$: We associate each size-k committee $U \subseteq C$, with two length $2^{2n}$ vectors with non-negative entries such that for all $U \subseteq C, t \in \mathcal{T}$,

$$f(U)[t] := |(C_t \cap U) \setminus W'| \text{ and } g(U)[t] := |(C_t \cap U) \cap W'|.$$

The vector $f(U)[t]$ is the number of alternatives of type $t$ in the committee without $W'$, and $g(U)[t]$ is the number of alternatives of type $t$ in the committee that are also in $W'$. For these pairs of vectors we define $\delta_c$-adjacency in the following way: Let $(f(W_1), g(W_1))$ and $(f(W_2), g(W_2))$ be the pairs of vectors representing two committees $W^1$ and $W^2$. We say that $(f(W_1), g(W_1))$ and $(f(W_2), g(W_2))$ are $\delta_c$-adjacent if $\sum_{t \in \mathcal{T}} |f(W_1)[t] - f(W_2)[t]| + |g(W_1)[t] - g(W_2)[t]| \leq 2\delta_c$.

It is clear there are at most $(k+1)^{2 \cdot 2^n}$ possible ordered pairs of vectors $f$ and $g$. For each such ordered pair of vectors we consider it *relevant* if there is a corresponding size-k committee, i.e., there are enough alternatives of each type in both $W'$ and $C \setminus W'$ and the score of such a committee is at least $\text{SC}(W) - \delta_s$. Note that the scores of any two committees with the same corresponding ordered pairs of vectors must be equal because we are considering a neutral voting rule. Let

$\hat{V} := \{(f(U), g(U)) \mid U \subseteq C, \text{sc}(W) - \delta_{\text{s}} \geq \text{sc}(U), |U| = \mathsf{k}\}$ be the set of ordered pairs of vectors that are relevant. We construct a restricted reconfiguration graph $\hat{G}$ whose vertices are $\hat{V}$, where there is an edge between two pairs of vectors if and only if they are $\delta_{\text{c}}$-adjacent. Such a graph takes $(\mathsf{k}+1)^{4 \cdot 2^n}(n+m)^{O(1)}$ time to construct.

We proceed to show that there is a $(\delta_{\text{c}}, \delta_{\text{s}})$-reconfiguration path of length $\ell'$ between $W$ and $W'$ if and only if there is a path in $\hat{G}$ between $(f(W), g(W))$ and $(f(W'), g(W'))$ of length at most $\ell'$.

Assume there is a $(\delta_{\text{c}}, \delta_{\text{s}})$-reconfiguration path $(W_0, \ldots, W_{\ell'})$, where $W_0 = W$ and $W_{\ell'} = W'$, of length $\ell'$. For every $W_j, j \in \{0, \ldots, \ell'\}$, we construct $(f(W_j), g(W_j))$ as previously described. We know that $(f(W_j), g(W_j))$ must be contained in $\hat{V}$ by construction. It is easy to observe that if $|W_j \Delta W_{j+1}| \leq 2\,\delta_{\text{c}}$, then $\sum_{t \in \mathcal{T}} |f(W_j)[t] - f(W_{j+1})[t]| + |g(W_j)[t] - g(W_{j+1})[t]| \leq 2\,\delta_{\text{c}}$ and the two vectors are $\delta_{\text{c}}$-adjacent.

Assume there is a path in $\hat{G}$ between $(f(W), g(W))$ and $(f(W'), g(W'))$ of length at most $\ell'$. We construct a reconfiguration path $(W_0, \ldots, W_{\ell'})$ such that $W_0 = W$, $W_{\ell'} = W'$ and for every $j \in \{0, \ldots, \ell'\}, t \in \mathcal{T}$, it holds that $f(W_j)[t] = |(\mathsf{C}_t \cap W_j) \setminus W'|$ and $g(W_j)[t] = |(\mathsf{C}_t \cap W_j) \cap W'|$.

We set $W_0 := W$. Assuming we have constructed committees $W_0, \ldots, W_j$ for some $j \in \{0, \ldots, \ell' - 1\}$ so that the above conditions are satisfied, we construct the next committee $W_{j+1}$ as follows:

(i) For every type $t \in \mathcal{T}$, if $f(W_{j+1})[t] - f(W_j)[t] > 0$, then we add arbitrary $f(W_{j+1})[t] - f(W_j)[t]$ many alternatives from $(\mathsf{C}_t \setminus W_j) \setminus W'$ to $W_j$.

(ii) For every type $t \in \mathcal{T}$, if $f(W_j)[t] - f(W_{j+1})[t] > 0$, then we remove arbitrary $f(W_j)[t] - f(W_{j+1})[t]$ many alternatives from $(W_j \cap \mathsf{C}_t) \setminus W'$ from $W_j$.

(iii) For every type $t \in \mathcal{T}$, if $g(W_{j+1})[t] - g(W_j)[t] > 0$, then we add arbitrary $g(W_{j+1})[t] - g(W_j)[t]$ many alternatives from $(\mathsf{C}_t \setminus W_j) \cap W'$ to $W_j$.

(iv) For every type $t \in \mathcal{T}$, if $g(W_j)[t] - g(W_{j+1})[t] > 0$, then we remove arbitrary $g(W_j)[t] - g(W_{j+1})[t]$ many alternatives from $(W_j \cap \mathsf{C}_t) \cap W'$ from $W_j$.

It is easy to see that if these exchanges can be performed, then $f(W_{j+1})[t] = |(\mathsf{C}_t \cap W_{j+1}) \setminus W'|$ and $g(W_{j+1})[t] = |(\mathsf{C}_t \cap W_{j+1}) \cap W'|$ must hold. We can perform Exchanges (i) and (iii), as per construction, there must be enough alternatives of the selected type $t$ in $(W_j \cap \mathsf{C}_t) \setminus W'$ and $(W_j \cap \mathsf{C}_t) \cap W'$. Similarly Exchanges (ii) and (iv) can be performed, because there must be sufficiently many alternatives in $(\mathsf{C}_t \setminus W') \setminus W_j$ and $(\mathsf{C}_t \cap W') \setminus W_j$, because otherwise $W_{j+1}$ could not exist in the reconfiguration path. Additionally, because $\vec{W}_j$ and $\vec{W}_{j+1}$ are $\delta_{\text{c}}$-adjacent, the committees $W_j$ and $W_{j+1}$ must also be $\delta_{\text{c}}$-adjacent, as the number of additions and removals in (i)–(iv) is upper-bounded by $\sum_{t \in \mathcal{T}} |f(W_j)[t] - f(W_{j+1})[t]| + |g(W_j)[t] - g(W_{j+1})[t]| \leq 2\,\delta_{\text{c}}$. The score of $W_{j+1}$ must be sufficient because otherwise $(f(W_{j+1}), g(W_{j+1}))$ would not be on the path.

To see that $W_{\ell'} = W'$, it is enough to observe that $W'$ is the only committee $U \subseteq C$ such that $f(U)[t] = |(\mathsf{C}_t \cap W') \setminus W'| = 0$ and $g(U)[t] = |(\mathsf{C}_t \cap W') \cap W'| = |\mathsf{C}_t \cap W'|$ for every $t \in \mathcal{T}$. Thus we have that $W_{\ell'} = W'$.

The FPT (resp. XP) statement follows directly from the time of constructing $\hat{G}$ and searching for a shortest path. $\qquad\square$

## B.5 Proof of Proposition 5

**Proposition 5** ($\star$). *For all neutral and score-based multi-winner voting rules $\lambda$ such that the score of a committee can be computed in polynomial time, $\lambda$-MR is solvable in $2^{2\mathsf{b}n}(n+m)^{O(1)}$ time, and hence FPT wrt. $n + \mathsf{b}$.*

*Proof.* Let $\mathcal{P} = (C = [m], V, R)$ be a preference profile and $I = (\mathcal{P}, \mathsf{k}, \delta_{\text{c}}, \delta_{\text{s}}, W, W')$ an instance of $\lambda$-MR. As every voter approves at most $\mathsf{b}$ alternatives, the number of alternatives that are approved by at least one voter is at most $\mathsf{b}n$. Let $\hat{C} = \cup_{i \in V} \mathsf{A}_i$ denote the set of alternatives that are approved by at least one voter. Then $|\hat{C}| \leq \mathsf{b}n$ We call these alternatives *relevant* and the rest of the alternatives *irrelevant*.

For every two size-$\mathsf{k}$ committees $W_1, W_2$ such that $W_1 \cap \hat{C} = W_2 \cap \hat{C}$, there is a reconfiguration path from $W_1$ to $W_2$ by swapping the irrelevant alternatives one by one. This is true because the irrelevant alternatives are not approved by any voter and $\lambda$ is neutral, so their identity does not affect

the score of the committee. We can use this observation to create a modified reconfiguration graph where we disregard the information concerning the identity of irrelevant alternatives.

We create a modified reconfiguration graph $\hat{G}^{\mathsf{R}}$. The vertices are all at most size-k committees of $\hat{C}$ such that, possibly together with some irrelevant alternatives, they could be committees on a reconfiguration path. Formally, the set of vertices of $\hat{G}^{\mathsf{R}}$ is

$$V(\hat{G}^{\mathsf{R}}) = \{u_{C'} \mid C' \subseteq \hat{C} \text{ s.t. } |C'| \le k \wedge \mathrm{sc}_\lambda(W) - \mathrm{sc}_\lambda(C' \cup \hat{W}) \le \delta_{\mathsf{s}}$$
$$\text{for some } \hat{W} \subseteq C \setminus \hat{C} \text{ with } |\hat{W}| = k - |C|\}.$$

Since $|\hat{C}| \le \mathsf{b}n$, we have that $|V(\hat{G}^{\mathsf{R}})| \le 2^{\mathsf{b}n}$. There is an edge between two vertices $u_{W_1}$ and $u_{W_2}$ if there are two committees containing $W_1$ and $W_2$ as their relevant alternatives, which are $\delta_{\mathsf{c}}$-adjacent. Formally $u_{W_1}$ and $u_{W_2}$ are adjacent if $|W_1 \setminus W_2| \le \delta_{\mathsf{c}}$ and $|W_2 \setminus W_1| \le \delta_{\mathsf{c}}$. The number of edges of $G^{\mathsf{R}}$, i.e., $|V(G^{\mathsf{R}})|^2$ is bounded above by $(2^{\mathsf{b}n})^2$. Similarly to the proof of Proposition 3, we can search for a path from $u_{W \cap \hat{C}}$ to $u_{W' \cap \hat{C}}$ in $2^{2\mathsf{b}n} \cdot (n + m)^{O(1)}$ time.

Finally, we show that there exists a path from $u_{W \cap \hat{C}}$ to $u_{W' \cap \hat{C}}$ in the modified reconfiguration graph if and only if there exists a $(\delta_{\mathsf{c}}, \delta_{\mathsf{s}})$-reconfiguration path for $I$.

For the if direction, let $W_0, \ldots, W_t$ with $W_0 = W$ and $W_t = W'$ be a reconfiguration path. For each committee $W_i$ on the path there must exist a vertex $u_{W_i \cap \hat{C}}$. Since $|W_i \setminus W_{i+1}| \le \delta_{\mathsf{c}}$ and $|W_{i+1} \setminus W_i| \le \delta_{\mathsf{c}}$ it follows that $|(W_{i+1} \cap \hat{C}) \setminus (W_i \cap \hat{C})| \le \delta_{\mathsf{c}}$ and $|(W_i \cap \hat{C}) \setminus (W_{i+1} \cap \hat{C})| \le \delta_{\mathsf{c}}$. Therefore it follows that $u_{W_1 \cap \hat{C}}, \ldots, u_{W_t \cap \hat{C}}$ is a path in the modified reconfiguration graph.

For the only-if direction, we first note that irrelevant alternatives can be exchanged with other irrelevant alternatives without changing the score. We order the irrelevant alternatives in an arbitrary but fixed way such that the irrelevant alternatives in $W$ are before all other irrelevant alternatives. Let $u_{V_0}, \ldots, u_{V_t}$ be a path in the reconfiguration graph such that $V_0 = W \cap \hat{C}$ and $V_t = W' \cap \hat{C}$. We will now iteratively build a reconfiguration path. We set $W_0 = W$. Let $W_0, \ldots, W_i$ be the committees corresponding to the first $i$ vertices on the path. We create $W_{i+1}$ in the following way:

If $|V_i| = |V_{i+1}|$, then we set $W_{i+1} = W_i \setminus (V_i \setminus V_{i+1}) \cup (V_{i+1} \setminus V_i)$. In this case $|W_{i+1}| = |W_i|$, $|W_{i+1} \setminus W_i| = |V_{i+1} \setminus V_i| \le \delta_{\mathsf{c}}$, and $|W_i \setminus W_{i+1}| = |V_i \setminus V_{i+1}| \le \delta_{\mathsf{c}}$. Therefore this is a valid reconfiguration step.

If $|V_i| > |V_{i+1}|$, then let $I$ be the $|V_i| - |V_{i+1}|$ many irrelevant alternatives with the lowest index which are not in $W_i$. We then set $W_{i+1} = W_i \setminus (V_i \setminus V_{i+1}) \cup (V_{i+1} \setminus V_i) \cup I$. In this case $|W_{i+1}| = |W_i|$ and due to this $|W_{i+1} \setminus W_i| = |W_i \setminus W_{i+1}|$. As $|W_i \setminus W_{i+1}| = |V_i \setminus V_{i+1}| \le \delta_{\mathsf{c}}$, this is also a valid reconfiguration step.

In the last case that $|V_i| < |V_{i+1}|$, let $I$ be the $|V_{i+1}| - |V_i|$ many irrelevant alternatives with the highest index which are in $W_i$. We then set $W_{i+1} = (W_i \setminus I) \setminus (V_i \setminus V_{i+1}) \cup (V_{i+1} \setminus V_i)$. In this case $|W_{i+1}| = |W_i|$ and due to this $|W_{i+1} \setminus W_i| = |W_i \setminus W_{i+1}|$. As $|W_i \setminus W_{i+1}| = |V_{i+1} \setminus V_i| \le \delta_{\mathsf{c}}$, this is also a valid reconfiguration step.

It is easy to see that the obtained $u_{V_{i+1}}$ is in the reconfiguration graph, as $\mathrm{sc}(V_{i+1} \cup (W_{i+1} \setminus \hat{C})) = \mathrm{sc}(W_{i+1}) \ge \mathrm{sc}(W) - \delta_{\mathsf{s}}$.

By repeating this process, we get a reconfiguration path from $W$ to a committee $W_t$ which satisfies $W_t \cap \hat{C} = W' \cap \hat{C}$. Therefore the alternatives in $W_t \Delta W'$ are only irrelevant alternatives. We can greedily exchange these alternatives to obtain a reconfiguration path from $W_t$ to $W'$, thereby completing our proof. □

## B.6 Proof of Proposition 6

**Proposition 6** (⋆)**.** DSR-TJ *is W[2]-hard wrt.* $h + \hat{\ell}$.

*Proof.* We provide a parameterized reduction from the W[2]-complete DOMINATING SET problem (Downey and Fellows, 2013) which is adapted from the one of Mouawad *et al.* (2016). We will point out the differences in the reduction.

DOMINATING SET (DS)
**Input:** A graph $\hat{G}$ and a non-negative integer $h$.
**Question:** Does $\hat{G}$ admit a dominating set of size $h$?

We first show the construction and then the correctness. Note that in our construction, the size of the dominating set as well as the length of a shortest reconfiguration path in the reduced instance will be linearly bounded by $h$.

**Construction.** Let $(\hat{G}, h)$ be a DS instance with $\hat{V} = \{u_1, \ldots, u_{\hat{n}}\}$ being the vertex set of $G$. The graph $G$ of the created instance of DSR-TJ consists of two independent graphs $H_1$ and $H_2$.

The first graph $H_1$ consists of $h + 4$ cliques (i.e., complete subgraphs), each of size $h + 3$ (rather than $h + 2$-cliques consisting of $h + 1$ vertices as in the original reduction of Mouawad et al.). We refer to these cliques as $C_0, \ldots, C_{h+3}$. We furthermore label the vertices in each clique $C_i$ as $v_{i,1}, \ldots, v_{i,h+3}$. Finally, we add to $H_1$ the following edges $E' = \{\{v_{0,i}, v_{z,i}\} \mid \forall i, z \in [h+3]\}$. In other words, we make each vertex in $C_0$ adjacent to the vertex in each other clique $C_z$ ($z \in [h+3]$) that have the same second index. Therefore, each vertex in each of the cliques $C_1, \ldots, C_{h+3}$ is adjacent to a vertex in $C_0$, with no two vertices from the same clique $C_z$ being adjacent to the same vertex in $C_0$.

The second graph $H_2$ is the same as the one in the original reduction. It consists of $h + 2$ copies of the original graph $\hat{G}$ which we label as $G_0 = (V_1, E_1), \ldots, G_{h+1} = (V_{h+1}, E_{h+1})$. For each $z \in \{0, \ldots, h+1\}$, let $w_{z,1}, \ldots, w_{z,|V|}$ denote the vertices in $G_i$. We add to $H_2$ the following edges $E'' = \{\{w_{0,i}, w_{z,j}\} \mid i, j \in [n], z \in [h+1] \text{ s.t. } \{u_i, u_j\} \in E(\hat{G}) \vee i = j\}$. In other words, we make each vertex $w_{0,i} \in G_0$ adjacent to the copies of its neighbors in $G_0$ and to its copies $w_{1,i}, \ldots, w_{h+1,i}$ in $G_1, \ldots, G_{h+1}$.

Finally, we add to $H_2$ a set $D$ of $h + 2$ dummy vertices $d_0, \ldots, d_{h+1}$ and edges that connect the copied graphs, one for each copy. For every $z \in \{0, \ldots, h+1\}$, we make $d_z$ adjacent to all vertices in $G_z$ and all vertices in $G_0$.

We define the initial and target dominating sets as $V_s = V(C_0) \cup D \cup \{w_{1,1}, \ldots, w_{1,h}\}$ and $V_t = \{v_{i,i} \mid i \in [h+3]\} \cup D \cup \{w_{1,1}, \ldots, w_{1,h}\}$, respectively. Note that the set $\{w_{1,1}, \ldots, w_{1,h}\}$ is not necessary to dominate the vertices in the created graph, but is needed to maintain the size bound; it can be replaced with an arbitrary but fixed vertex subset of $V(G_0)$ of size $h$. We use this subset for ease of reasoning.

To complete the construction, we define the size bound $\hat{h} = 3h + 5$ and $\ell = 4h + 5$. Clearly, the construction can be done in polynomial time and $|V_s| = |V_t| = \hat{h}$.

**Correctness.** We show that $(G, h)$ admits a size-$h$ dominating set if and only if the constructed instance has a length-$\ell$ reconfiguration path between $V_s$ and $V_t$. For the "only if" part, let $U'$ denote a size-$h$ dominating set of $\hat{G}$. We first exchange the $h$ vertices $w_{1,z}$, $z \in [h]$, one-by-one with the vertices from the copies corresponding to $V'$, i.e., $\{w_{1,i} \mid u_i \in U'\}$. Afterwards, we exchange each dummy vertex from $D$ with a distinct vertex $v_{z,z}$, $z \in [h+2]$. Next, we exchange one vertex from $V(C_0)$ with vertex $v_{h+3,h+3}$. Note that now the remaining vertices from the clique $C_0$ are not necessary to dominate the vertices in the created graph anymore. Thus, we can exchange the remaining $h + 2$ vertices from the clique $C_0$ with the $h + 2$ dummies from $D$, one-by-one, reaching the target dominating set. The length of the reconfiguration path is $h + h + 2 + h + 3 = 3h + 5 = \ell$.

For the "if" part, assume that the constructed instance admits a length-$\ell$ reconfiguration path between $V_s$ and $V_t$. We aim to show that on the reconfiguration path there must be a dominating set of $H_1 \cup H_2$ containing a size-$h$ subset which corresponds to a dominating set of $\hat{G}$. We recall an observation by Mouawad et al. that for $h + 4$ cliques, each of size $h + 3$, and connected as described in their construction, they need $h + 3$ additional vertices on that graph in order to be able to switch from the dominating set consisting of $V(C_0)$ to the dominating set $\{v_{i,i} \mid i \in [h+2]\}$ via TAR, as one needs to dominate all vertices from $C_z$, $z \in [h+3]$, before removing any vertex from $C_0$.

We show similarly that for the TJ variant we can only afford to have $h$ vertices to dominate all vertices from $H_2$ in order to "save up" $h + 2$ to exchange for the vertices from $\{v_{z,z} \mid z \in [h+2]\}$. To achieve this, we claim that for each dominating set $X$ on the reconfiguration path which does not contain all vertices from the clique $C_0$, i.e., $|X \cap V(C_0)| \neq |V(C_0)|$, it must hold that $|V(C_z) \in X| \geq 1$ for every $z \in [h+2]$. Let $v_{0,i}$ be a vertex that is not included in $X$. Then, in order to dominate all vertices

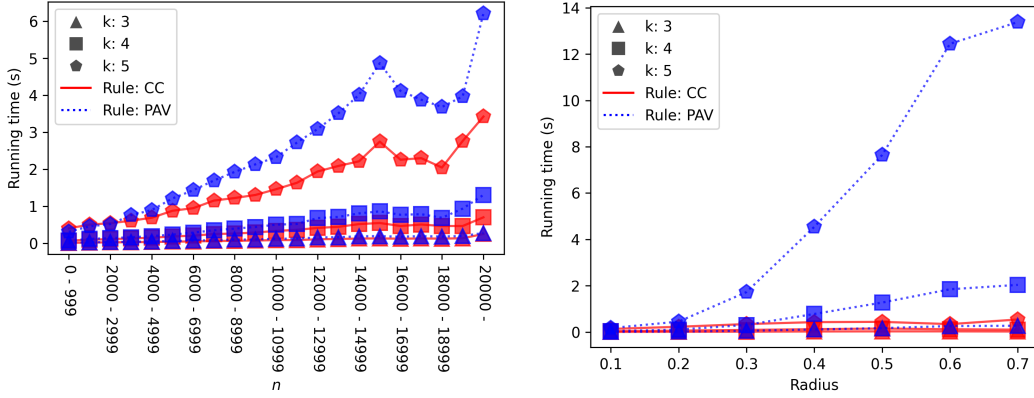

Figure 3: Left: The dependency of the median time to find a reconfiguration path to the number of voters $n$ on Netflix data. Right: The dependency of the median time to find a reconfiguration path to the radius of Manhattan data.

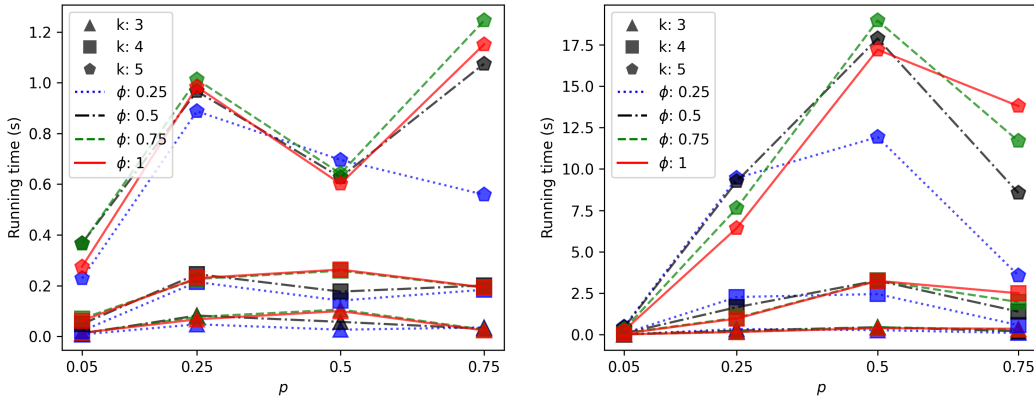

Figure 4: Left: CC, Right: PAV. The dependency of a median time to find a reconfiguration path to $p$ under $(p\text{-}\phi)$-Resampling data, measured in seconds.

from $\{v_{z,i} \mid z \in [h+3]\}$, we need a vertex from each clique $C_z$, as claimed. Since no vertex in $C_0$ is contained in $V_t$, in order to reconfigure from $V_s$ to $V_t$, there must be an intermediate dominating set $X'$ where one vertex from $V(C_0)$ is missing and at least one vertex from each clique $C_z$ is present. This means that $X'$ contains $h + 3 + h + 2 = 2h + 5$ vertices from the first graph $H_1$. By the size bound $\hat{h} = 3h + 5$, we need $h$ vertices to dominate the second graph $H_2$. By construction, this is possible only if it corresponds to a size-$h$ dominating set of the input graph $\hat{G}$. This concludes the "if" part and the correctness proof. □

## C   Experimental Research

In this section, we present some preliminary research on how easy reconfiguration paths are to find in practice. We discover that reconfiguration paths almost always exist and can be found quite efficiently. We use both real-life (Netflix Prize data (Netflix, 2006)) and synthetic data (geometric and $(p\text{-}\phi)$-resampling).

### C.1   Setup and Heuristic

We perform the experiments for both CC and PAV  and committee sizes $\mathsf{k} \in \{3, 4, 5\}$.

For each instance-rule-$\mathsf{k}$-combination, we create 100 random committees. We order them by score, then pick and pair the 20 best ones. This way we obtain random but still reasonably well-scoring

Table 2: Netflix data: Median, average, and variance of the running time, and the number of committee pairs. The data is only taken on the committee pairs that did not time out, except the total number of committee pairs. The proportion of timeouts is in Table 3. Timeout 60s.

| Rule | n | k = 3 | | | | k = 4 | | | | k = 5 | | | |
|---|---|---|---|---|---|---|---|---|---|---|---|---|---|
| | | Med. | Avg. | $\sigma^2$ | # Pairs | Med. | Avg. | $\sigma^2$ | # Pairs | Med. | Avg. | $\sigma^2$ | # Pairs |
| CC | 0 - 999 | 0.01 | 0.02 | 0.00 | 9020 | 0.07 | 0.07 | 0.00 | 9020 | 0.40 | 0.41 | 0.04 | 9020 |
| | 1000 - 1999 | 0.02 | 0.03 | 0.00 | 2650 | 0.10 | 0.10 | 0.00 | 2650 | 0.51 | 0.50 | 0.05 | 2650 |
| | 2000 - 2999 | 0.03 | 0.03 | 0.00 | 980 | 0.11 | 0.11 | 0.00 | 980 | 0.54 | 0.52 | 0.05 | 980 |
| | 3000 - 3999 | 0.03 | 0.04 | 0.00 | 1190 | 0.13 | 0.14 | 0.00 | 1190 | 0.61 | 0.63 | 0.07 | 1190 |
| | 4000 - 4999 | 0.04 | 0.04 | 0.00 | 1170 | 0.15 | 0.16 | 0.00 | 1170 | 0.69 | 0.72 | 0.09 | 1170 |
| | 5000 - 5999 | 0.04 | 0.05 | 0.00 | 720 | 0.19 | 0.20 | 0.01 | 720 | 0.88 | 0.94 | 0.16 | 720 |
| | 6000 - 6999 | 0.05 | 0.06 | 0.00 | 680 | 0.21 | 0.23 | 0.01 | 680 | 0.95 | 1.04 | 0.24 | 680 |
| | 7000 - 7999 | 0.06 | 0.08 | 0.00 | 830 | 0.25 | 0.28 | 0.02 | 830 | 1.15 | 1.31 | 1.01 | 830 |
| | 8000 - 8999 | 0.07 | 0.08 | 0.00 | 860 | 0.26 | 0.29 | 0.02 | 860 | 1.22 | 1.40 | 0.61 | 860 |
| | 9000 - 9999 | 0.08 | 0.09 | 0.00 | 590 | 0.29 | 0.31 | 0.02 | 600 | 1.31 | 1.52 | 1.70 | 600 |
| | 10000 - 10999 | 0.09 | 0.12 | 0.04 | 560 | 0.33 | 0.42 | 0.56 | 560 | 1.46 | 1.93 | 9.34 | 560 |
| | 11000 - 11999 | 0.09 | 0.11 | 0.01 | 430 | 0.36 | 0.40 | 0.04 | 430 | 1.63 | 1.80 | 0.86 | 430 |
| | 12000 - 12999 | 0.11 | 0.18 | 0.21 | 430 | 0.42 | 0.56 | 1.04 | 430 | 1.94 | 2.34 | 7.09 | 430 |
| | 13000 - 13999 | 0.12 | 0.14 | 0.01 | 340 | 0.45 | 0.49 | 0.05 | 350 | 2.09 | 2.18 | 1.18 | 350 |
| | 14000 - 14999 | 0.13 | 0.15 | 0.01 | 220 | 0.52 | 0.57 | 0.07 | 220 | 2.22 | 2.34 | 1.42 | 220 |
| | 15000 - 15999 | 0.13 | 0.16 | 0.01 | 300 | 0.55 | 0.60 | 0.08 | 300 | 2.75 | 2.76 | 2.05 | 300 |
| | 16000 - 16999 | 0.13 | 0.14 | 0.00 | 190 | 0.47 | 0.51 | 0.06 | 190 | 2.26 | 2.37 | 1.45 | 190 |
| | 17000 - 17999 | 0.13 | 0.32 | 1.50 | 220 | 0.51 | 0.99 | 6.14 | 220 | 2.30 | 3.21 | 29.32 | 220 |
| | 18000 - 18999 | 0.13 | 0.21 | 0.11 | 100 | 0.47 | 0.53 | 0.09 | 100 | 2.05 | 2.27 | 1.02 | 100 |
| | 19000 - 19999 | 0.14 | 0.16 | 0.01 | 20 | 0.47 | 0.49 | 0.02 | 20 | 2.76 | 2.96 | 0.91 | 20 |
| | 20000 - | 0.25 | 0.32 | 0.06 | 30 | 0.70 | 1.21 | 2.44 | 30 | 3.42 | 5.31 | 23.19 | 30 |
| PAV | 0 - 999 | 0.02 | 0.02 | 0.00 | 9020 | 0.07 | 0.08 | 0.00 | 9020 | 0.31 | 0.41 | 0.12 | 9020 |
| | 1000 - 1999 | 0.03 | 0.03 | 0.00 | 2650 | 0.12 | 0.13 | 0.02 | 2650 | 0.46 | 0.57 | 0.41 | 2650 |
| | 2000 - 2999 | 0.03 | 0.04 | 0.00 | 980 | 0.14 | 0.14 | 0.00 | 980 | 0.52 | 0.60 | 0.10 | 980 |
| | 3000 - 3999 | 0.04 | 0.05 | 0.00 | 1190 | 0.16 | 0.18 | 0.01 | 1190 | 0.76 | 0.83 | 0.23 | 1190 |
| | 4000 - 4999 | 0.05 | 0.06 | 0.00 | 1170 | 0.19 | 0.21 | 0.01 | 1170 | 0.90 | 0.99 | 0.37 | 1170 |
| | 5000 - 5999 | 0.06 | 0.07 | 0.00 | 720 | 0.24 | 0.27 | 0.02 | 720 | 1.21 | 1.39 | 0.66 | 720 |
| | 6000 - 6999 | 0.07 | 0.08 | 0.00 | 680 | 0.29 | 0.33 | 0.03 | 680 | 1.43 | 1.63 | 0.89 | 680 |
| | 7000 - 7999 | 0.09 | 0.10 | 0.00 | 830 | 0.35 | 0.43 | 0.09 | 830 | 1.70 | 2.10 | 2.24 | 830 |
| | 8000 - 8999 | 0.10 | 0.12 | 0.01 | 860 | 0.40 | 0.48 | 0.12 | 860 | 1.93 | 2.39 | 3.19 | 860 |
| | 9000 - 9999 | 0.10 | 0.13 | 0.02 | 590 | 0.44 | 0.53 | 0.13 | 600 | 2.13 | 2.58 | 4.40 | 600 |
| | 10000 - 10999 | 0.12 | 0.17 | 0.05 | 560 | 0.50 | 0.71 | 2.01 | 560 | 2.33 | 3.07 | 17.26 | 560 |
| | 11000 - 11999 | 0.13 | 0.16 | 0.02 | 430 | 0.52 | 0.63 | 0.15 | 430 | 2.72 | 3.16 | 3.64 | 430 |
| | 12000 - 12999 | 0.16 | 0.26 | 0.22 | 430 | 0.68 | 1.05 | 6.15 | 430 | 3.09 | 3.77 | 9.49 | 430 |
| | 13000 - 13999 | 0.17 | 0.20 | 0.02 | 340 | 0.71 | 0.84 | 0.27 | 350 | 3.51 | 4.02 | 6.83 | 350 |
| | 14000 - 14999 | 0.19 | 0.24 | 0.03 | 220 | 0.81 | 0.95 | 0.35 | 220 | 4.01 | 5.12 | 14.54 | 220 |
| | 15000 - 15999 | 0.19 | 0.24 | 0.03 | 300 | 0.86 | 1.02 | 0.37 | 300 | 4.86 | 5.59 | 13.46 | 300 |
| | 16000 - 16999 | 0.18 | 0.22 | 0.02 | 190 | 0.77 | 0.90 | 0.28 | 190 | 4.12 | 4.71 | 10.21 | 190 |
| | 17000 - 17999 | 0.20 | 0.64 | 8.75 | 220 | 0.79 | 1.99 | 36.48 | 220 | 3.88 | 5.41 | 38.97 | 220 |
| | 18000 - 18999 | 0.18 | 0.28 | 0.07 | 100 | 0.69 | 0.86 | 0.32 | 100 | 3.68 | 4.10 | 7.14 | 100 |
| | 19000 - 19999 | 0.20 | 0.21 | 0.01 | 20 | 0.94 | 1.00 | 0.20 | 20 | 3.97 | 4.31 | 6.65 | 20 |
| | 20000 - | 0.28 | 0.39 | 0.08 | 30 | 1.31 | 2.65 | 7.23 | 30 | 6.21 | 10.69 | 102.87 | 30 |

committees. If the number of alternatives is too small to obtain 100 random committees, we skip the instance. For each pair, we choose the lower scoring committee as the starting committee $W$, the higher scoring as the goal committee $W'$, and set $\delta_s = 0$ and $\delta_c = 1$.

To find a reconfiguration path, we create an initial set $S := W \cup W'$. We create the reconfiguration graph that only uses the alternatives in $S$. We use Python and our modification of the abcvoting library (Lackner *et al.*, 2023) to find all the committees that use alternatives in $S$ and whose score is at least $\text{SC}(W)$. If there is no $W$-$W'$ path in this graph, we add the alternative with the highest number of approvals outside of $S$ to $S$, and add the sufficiently high-scoring committees within $S$ that contain this alternative to the reconfiguration graph. We repeat this until we either find a reconfiguration path, show there is no path, or time out.

The experiments are run on Python 3.10.12. Additionally we use for example the following Python libraries: abcvoting version 2.8.0, Gurobipy version 10.0.2 and Networkx version 3.1. A complete list of all the dependencies is in the supplementary material. We modify the abcvoting library to be able to compute all committees whose score is at least some percentage of the optimal score. The experiments are run on Ubuntu 22.04.2 with 3.8 GB of RAM, CPU of the virtual machine "Intel Xeon (Cascadelake), 2x 1-Core", on a host with "Intel Xeon Gold 5215, 2x 10-Core".

We set the timeout for finding the reconfiguration path to 60 seconds.

Table 3: Netflix data: The proportion of timeout pairs. Timeout 60s.

| | k = 3 | | k = 4 | | k = 5 | |
|---|---|---|---|---|---|---|
| $n$ | CC | PAV | CC | PAV | CC | PAV |
| 0 - 999 | 0.0 % | 0.0 % | 0.0 % | 0.0 % | 0.0 % | 0.0 % |
| 1000 - 1999 | 0.0 % | 0.0 % | 0.0 % | 0.0 % | 0.0 % | 0.0 % |
| 2000 - 2999 | 0.0 % | 0.0 % | 0.0 % | 0.0 % | 0.0 % | 0.0 % |
| 3000 - 3999 | 0.0 % | 0.0 % | 0.0 % | 0.0 % | 0.0 % | 0.0 % |
| 4000 - 4999 | 0.0 % | 0.0 % | 0.0 % | 0.0 % | 0.0 % | 0.0 % |
| 5000 - 5999 | 0.0 % | 0.0 % | 0.0 % | 0.0 % | 0.0 % | 0.0 % |
| 6000 - 6999 | 0.0 % | 0.0 % | 0.0 % | 0.0 % | 0.0 % | 0.0 % |
| 7000 - 7999 | 0.0 % | 0.0 % | 0.0 % | 0.0 % | 0.0 % | 0.0 % |
| 8000 - 8999 | 0.0 % | 0.0 % | 0.0 % | 0.0 % | 0.0 % | 0.0 % |
| 9000 - 9999 | 0.0 % | 0.0 % | 0.0 % | 0.0 % | 0.0 % | 0.0 % |
| 10000 - 10999 | 0.0 % | 0.0 % | 0.0 % | 0.0 % | 0.0 % | 0.2 % |
| 11000 - 11999 | 0.0 % | 0.0 % | 0.0 % | 0.0 % | 0.0 % | 0.0 % |
| 12000 - 12999 | 0.0 % | 0.0 % | 0.0 % | 0.0 % | 0.0 % | 0.2 % |
| 13000 - 13999 | 0.0 % | 0.0 % | 0.0 % | 0.0 % | 0.0 % | 0.0 % |
| 14000 - 14999 | 0.0 % | 0.0 % | 0.0 % | 0.0 % | 0.0 % | 0.0 % |
| 15000 - 15999 | 0.0 % | 0.0 % | 0.0 % | 0.0 % | 0.0 % | 0.0 % |
| 16000 - 16999 | 0.0 % | 0.0 % | 0.0 % | 0.0 % | 0.0 % | 0.0 % |
| 17000 - 17999 | 0.0 % | 0.0 % | 0.0 % | 0.5 % | 1.4 % | 1.8 % |
| 18000 - 18999 | 0.0 % | 0.0 % | 0.0 % | 0.0 % | 0.0 % | 0.0 % |
| 19000 - 19999 | 0.0 % | 0.0 % | 0.0 % | 0.0 % | 0.0 % | 0.0 % |
| 20000 - | 0.0 % | 0.0 % | 0.0 % | 0.0 % | 0.0 % | 0.0 % |

Table 4: Manhattan data: Median, average, and variance of the running time. The data is only taken on the committee pairs that did not time out. The proportion of timeouts is in Table 5. Timeout 60s. There are 100 instances for each radius.

| | | k = 3 | | | k = 4 | | | k = 5 | | |
|---|---|---|---|---|---|---|---|---|---|---|
| Rule | Radius | Med. | Avg. | $\sigma^2$ | Med. | Avg. | $\sigma^2$ | Med. | Avg. | $\sigma^2$ |
| | 0.1 | 0.01 | 0.01 | 0.00 | 0.04 | 0.03 | 0.00 | 0.14 | 0.14 | 0.01 |
| | 0.2 | 0.01 | 0.02 | 0.02 | 0.06 | 0.08 | 0.15 | 0.23 | 0.27 | 0.03 |
| | 0.3 | 0.02 | 0.04 | 0.04 | 0.09 | 0.11 | 0.01 | 0.35 | 0.40 | 0.07 |
| CC | 0.4 | 0.03 | 0.16 | 0.77 | 0.13 | 0.22 | 1.05 | 0.43 | 0.51 | 1.19 |
| | 0.5 | 0.03 | 0.32 | 2.02 | 0.15 | 0.31 | 2.74 | 0.44 | 0.50 | 0.09 |
| | 0.6 | 0.03 | 0.32 | 4.80 | 0.12 | 0.18 | 1.88 | 0.34 | 0.40 | 0.07 |
| | 0.7 | 0.02 | 0.14 | 2.27 | 0.10 | 0.10 | 0.00 | 0.54 | 0.52 | 0.06 |
| | 0.1 | 0.01 | 0.01 | 0.00 | 0.04 | 0.04 | 0.00 | 0.17 | 0.16 | 0.01 |
| | 0.2 | 0.02 | 0.03 | 0.00 | 0.11 | 0.11 | 0.00 | 0.45 | 0.52 | 0.14 |
| | 0.3 | 0.06 | 0.06 | 0.00 | 0.31 | 0.35 | 0.22 | 1.73 | 1.92 | 1.61 |
| PAV | 0.4 | 0.11 | 0.12 | 0.05 | 0.78 | 0.79 | 0.18 | 4.54 | 4.89 | 8.87 |
| | 0.5 | 0.17 | 0.20 | 0.14 | 1.28 | 1.26 | 0.36 | 7.64 | 7.86 | 19.70 |
| | 0.6 | 0.26 | 0.30 | 1.65 | 1.85 | 1.78 | 0.63 | 12.44 | 11.73 | 33.98 |
| | 0.7 | 0.27 | 0.28 | 0.02 | 2.04 | 1.98 | 0.68 | 13.38 | 12.12 | 34.25 |

Table 5: Manhattan data: The proportion of timeout pairs. Timeout 60s.

| | k = 3 | | k = 4 | | k = 5 | |
|---|---|---|---|---|---|---|
| Radius | CC | PAV | CC | PAV | CC | PAV |
| 0.1 | 0.0 % | 0.0 % | 0.0 % | 0.0 % | 0.0 % | 0.0 % |
| 0.2 | 0.0 % | 0.0 % | 0.2 % | 0.0 % | 0.0 % | 0.0 % |
| 0.3 | 0.0 % | 0.0 % | 0.2 % | 0.0 % | 0.0 % | 0.0 % |
| 0.4 | 0.0 % | 0.0 % | 0.0 % | 0.1 % | 0.1 % | 0.0 % |
| 0.5 | 0.0 % | 0.0 % | 0.1 % | 0.1 % | 0.1 % | 0.0 % |
| 0.6 | 0.2 % | 0.0 % | 0.1 % | 0.0 % | 0.0 % | 0.0 % |
| 0.7 | 0.0 % | 0.0 % | 0.0 % | 0.0 % | 0.0 % | 0.0 % |

Table 6: Resampling data: Median, average, and variance of the running time. The data is only taken on the committee pairs that did not time out. The proportion of timeouts is in Table 7. Timeout 60s. There are 100 instances for each $(p$-$\phi)$-pair.

| Rule | $p$ | $\phi$ | k = 3 | | | k = 4 | | | k = 5 | | |
|---|---|---|---|---|---|---|---|---|---|---|---|
| | | | Med. | Avg. | $\sigma^2$ | Med. | Avg. | $\sigma^2$ | Med. | Avg. | $\sigma^2$ |
| CC | 0.05 | 0.25 | 0.01 | 0.02 | 0.00 | 0.02 | 0.09 | 0.01 | 0.23 | 0.42 | 0.18 |
| | | 0.5 | 0.01 | 0.02 | 0.00 | 0.05 | 0.07 | 0.00 | 0.37 | 0.42 | 0.13 |
| | | 0.75 | 0.01 | 0.02 | 0.00 | 0.07 | 0.06 | 0.00 | 0.37 | 0.34 | 0.06 |
| | | 1 | 0.01 | 0.02 | 0.00 | 0.06 | 0.06 | 0.00 | 0.28 | 0.30 | 0.03 |
| | 0.25 | 0.25 | 0.05 | 0.06 | 0.00 | 0.21 | 0.21 | 0.01 | 0.89 | 0.95 | 0.40 |
| | | 0.5 | 0.08 | 0.08 | 0.00 | 0.25 | 0.24 | 0.01 | 0.97 | 0.98 | 0.30 |
| | | 0.75 | 0.08 | 0.07 | 0.00 | 0.23 | 0.22 | 0.01 | 1.01 | 0.98 | 0.25 |
| | | 1 | 0.07 | 0.23 | 2.59 | 0.23 | 0.28 | 0.67 | 0.98 | 0.98 | 0.30 |
| | 0.5 | 0.25 | 0.03 | 0.03 | 0.00 | 0.14 | 0.14 | 0.00 | 0.70 | 0.81 | 0.22 |
| | | 0.5 | 0.06 | 0.13 | 0.62 | 0.18 | 0.22 | 0.02 | 0.63 | 0.74 | 0.18 |
| | | 0.75 | 0.11 | 0.20 | 0.72 | 0.26 | 0.44 | 5.82 | 0.64 | 0.77 | 0.28 |
| | | 1 | 0.10 | 1.15 | 22.22 | 0.26 | 1.06 | 24.11 | 0.60 | 0.79 | 5.25 |
| | 0.75 | 0.25 | 0.04 | 0.03 | 0.00 | 0.18 | 0.15 | 0.00 | 0.56 | 0.82 | 0.20 |
| | | 0.5 | 0.03 | 0.03 | 0.00 | 0.20 | 0.17 | 0.01 | 1.07 | 0.94 | 0.23 |
| | | 0.75 | 0.03 | 0.09 | 1.47 | 0.19 | 0.17 | 0.00 | 1.24 | 1.00 | 0.22 |
| | | 1 | 0.02 | 0.14 | 0.31 | 0.19 | 0.18 | 0.01 | 1.15 | 0.96 | 0.20 |
| PAV | 0.05 | 0.25 | 0.01 | 0.04 | 0.00 | 0.07 | 0.17 | 0.08 | 0.48 | 0.92 | 2.61 |
| | | 0.5 | 0.02 | 0.03 | 0.00 | 0.08 | 0.10 | 0.01 | 0.42 | 0.55 | 0.31 |
| | | 0.75 | 0.02 | 0.02 | 0.00 | 0.09 | 0.08 | 0.00 | 0.32 | 0.32 | 0.06 |
| | | 1 | 0.02 | 0.02 | 0.00 | 0.07 | 0.07 | 0.00 | 0.25 | 0.25 | 0.02 |
| | 0.25 | 0.25 | 0.34 | 0.39 | 0.10 | 2.27 | 2.09 | 1.27 | 9.45 | 9.34 | 28.37 |
| | | 0.5 | 0.24 | 0.27 | 0.04 | 1.66 | 1.71 | 1.05 | 9.25 | 10.37 | 43.26 |
| | | 0.75 | 0.17 | 0.18 | 0.01 | 1.02 | 1.07 | 0.39 | 7.64 | 7.60 | 22.35 |
| | | 1 | 0.16 | 0.16 | 0.01 | 0.97 | 0.96 | 0.55 | 6.43 | 5.99 | 9.71 |
| | 0.5 | 0.25 | 0.27 | 0.35 | 0.07 | 2.45 | 2.34 | 1.42 | 11.92 | 10.62 | 24.53 |
| | | 0.5 | 0.45 | 0.43 | 0.05 | 3.26 | 3.00 | 1.62 | 17.87 | 15.87 | 44.97 |
| | | 0.75 | 0.45 | 0.44 | 0.05 | 3.26 | 3.02 | 1.41 | 18.95 | 16.69 | 46.51 |
| | | 1 | 0.39 | 0.39 | 0.16 | 3.25 | 2.99 | 1.57 | 17.20 | 15.72 | 58.93 |
| | 0.75 | 0.25 | 0.10 | 0.10 | 0.00 | 0.57 | 0.56 | 0.05 | 3.57 | 3.56 | 5.53 |
| | | 0.5 | 0.21 | 0.20 | 0.01 | 1.39 | 1.30 | 0.24 | 8.55 | 7.61 | 10.59 |
| | | 0.75 | 0.29 | 0.29 | 0.02 | 1.97 | 1.83 | 0.49 | 11.68 | 10.33 | 19.17 |
| | | 1 | 0.36 | 0.34 | 0.03 | 2.50 | 2.30 | 0.85 | 13.80 | 12.06 | 26.80 |

Table 7: Resampling data: The proportion of timeout pairs. Timeout 60s.

| $p$ | $\phi$ | k = 3 | | k = 4 | | k = 5 | |
|---|---|---|---|---|---|---|---|
| | | CC | PAV | CC | PAV | CC | PAV |
| 0.05 | 0.25 | 0.0 % | 0.0 % | 0.0 % | 0.0 % | 0.0 % | 0.0 % |
| | 0.5 | 0.0 % | 0.0 % | 0.0 % | 0.0 % | 0.0 % | 0.0 % |
| | 0.75 | 0.0 % | 0.0 % | 0.0 % | 0.0 % | 0.0 % | 0.0 % |
| | 1 | 0.0 % | 0.0 % | 0.0 % | 0.0 % | 0.0 % | 0.0 % |
| 0.25 | 0.25 | 0.0 % | 0.0 % | 0.0 % | 0.0 % | 0.0 % | 0.0 % |
| | 0.5 | 0.0 % | 0.0 % | 0.0 % | 0.0 % | 0.0 % | 0.0 % |
| | 0.75 | 0.0 % | 0.0 % | 0.0 % | 0.0 % | 0.0 % | 0.0 % |
| | 1 | 0.1 % | 0.0 % | 0.4 % | 0.0 % | 0.4 % | 0.0 % |
| 0.5 | 0.25 | 0.0 % | 0.0 % | 0.0 % | 0.0 % | 0.0 % | 0.0 % |
| | 0.5 | 0.0 % | 0.0 % | 0.0 % | 0.0 % | 0.0 % | 0.0 % |
| | 0.75 | 0.0 % | 0.0 % | 0.8 % | 0.0 % | 0.1 % | 0.0 % |
| | 1 | 0.3 % | 0.1 % | 1.6 % | 0.1 % | 0.8 % | 0.0 % |
| 0.75 | 0.25 | 0.0 % | 0.0 % | 0.0 % | 0.0 % | 0.0 % | 0.0 % |
| | 0.5 | 0.0 % | 0.0 % | 0.0 % | 0.0 % | 0.0 % | 0.0 % |
| | 0.75 | 0.0 % | 0.0 % | 0.0 % | 0.0 % | 0.0 % | 0.0 % |
| | 1 | 0.0 % | 0.1 % | 0.0 % | 0.0 % | 0.0 % | 0.0 % |

### C.1.1 Data Preprocessing

**Netflix data.** Netflix data contains ratings for 17770 movies from 480189 users. The ratings contain the date of the rating and an integer score from the range $[1, 5]$. To create approval preferences, we interpret a rating lower than 2 or a lack of a rating as disapproval, and a rating of at least 3 as approval. To make multiple elections, we divide the ratings by date. There are in total 2180 days.

We turn the ratings into approval votes and divide them by dates. To make the instances smaller, for each day, we remove all the movies that got fewer than $5\%$ of the number of approvals of the most popular movie of the day. We also ignore each day with fewer than 10 voters (i.e., users) or 5 alternatives (i.e., movies). In the end 2162 instances remain.

**Synthetic data.** In all synthetic instances, we have $n = 100$ voters and $m = 100$ alternatives. However, if either a voter does not approve of any alternatives or an alternative is approved by no voter, then it is removed from the instance. As a result, some instances have fewer voters or alternatives. We create two different types of data sets.

The first type uses geometric (Manhattan) preferences: We choose the locations of both voters and alternatives uniformly at random in $[0, 1] \times [0, 1]$. We fix a radius $r$, and each voter approves the alternatives that are within distance $r$ from him according to the metric given by the $\ell_1$-norm. We use the values $\{0.1, 0.2, \ldots, 0.7\}$ for $r$ and create 100 instances for every value of $r$.

The second type uses $(p\text{-}\phi)$-resampling model (Szufa *et al.*, 2022). In this model the variable $p$ is the probability that a voter approves of an alternative, and $\phi \in [0, 1]$ controls how dissimilar the voters are to each other; the higher $\phi$, the less similar the voters are. We use values $(p, \phi) \in \{0.05, 0.25, 0.5, 0.75\} \times \{0.25, 0.5, 0.75, 1\}$, and create 100 instances for every combination.

More formally, we first create a central vote that consists of $\lfloor p \cdot m \rfloor$ randomly selected alternatives. The central vote disapproves all other alternatives. For every voter-alternative pair $(v_i, c_i)$, the voter $v_i$ takes his opinion on $c_i$ from the central vote with probability $1 - \phi$ and with probability $\phi$ "resamples" his opinion on $c_i$: He approves $c_i$ with probability $p$.

### C.2 Evaluation

The median running times of the instances that did not time out are presented in Figure 3 and Figure 4. More detailed data is presented in Tables 2 to 7.

Overall we discover that finding a reconfiguration path is relatively fast in practice, and CC is noticeably faster than PAV. Despite our timeout being only 60 seconds, less than 0.03% of the committee pairs time out. More detailed breakdown of timeouts can be found in Tables 3, 5 and 7.

Of the pairs that did not time out, all but three pairs of committees admit a reconfiguration path. The three pairs in question were all from Manhattan data with radius 0.6 and k = 3. We must however note that it is rather difficult to show non-existence of reconfiguration path, because it requires in theory creating the whole reconfiguration graph, whose size is in $O(m^k)$.

Median running times for Netflix data, divided by the number of voters, are presented in Figure 3[Left]. More detailed information is available in Tables 2 and 3. The effect of the number of voters is surprisingly small.

Median running times of Manhattan data, divided by different values of $p$ and $\phi$, are presented in Figure 3[Right], and more detailed information in Tables 4 and 5. The larger the radius, i.e., the more alternatives the voters approve, the harder PAV appears to be.

Median running times of $(p\text{-}\phi)$-resampling data, divided by different values of $p$ and $\phi$, are presented in Figure 4, and more detailed information in Tables 6 and 7. The effect of $\phi$ to the running time appear to be rather moderate, although we can see from Table 7 that the timeouts are most frequent when $\phi$ is high, i.e., the votes are dissimilar. We see from Figure 4[Right] that under PAV the approval probability $p$ has a clear effect on the running time: $p = 0.5$ has the highest running time, with the times decreasing further away. The problem appears easier when the alternatives either approve a few or almost all alternatives.

